# Disentangling Linear Quadratic Control with Untrusted ML Predictions

**Tongxin Li**[1*]     **Hao Liu**[2*]     **Yisong Yue**[2]

[1]School of Data Science
The Chinese University of Hong Kong, Shenzhen, China
`litongxin@cuhk.edu.cn`

[2]Computing + Mathematical Sciences
California Institute of Technology, Pasadena, USA
`{hliu3, yyue}@caltech.edu`

## Abstract

Uncertain perturbations in dynamical systems often arise from diverse resources, represented by latent components. The predictions for these components, typically generated by "black-box" machine learning tools, are prone to inaccuracies. To tackle this challenge, we introduce DISC, a novel policy that learns a confidence parameter online to harness the potential of accurate predictions while also mitigating the impact of erroneous forecasts. When predictions are precise, DISC leverages this information to achieve near-optimal performance. Conversely, in the case of significant prediction errors, it still has a worst-case competitive ratio guarantee. We provide competitive ratio bounds for DISC under both linear mixing of latent variables as well as a broader class of mixing functions. Our results highlight a first-of-its-kind "best-of-both-worlds" integration of machine-learned predictions, thus lead to a near-optimal consistency and robustness tradeoff, which provably improves what can be obtained without learning the confidence parameter. We validate the applicability of DISC across a spectrum of practical scenarios.

## 1 Introduction

We study the problem of online decision-making with predictions. Such settings are increasingly popular with the use of machine learning for predictive modeling, with applications to power grids and robotics [1, 2, 3, 4]. However, in many real-world settings, such black-box predictors can be unreliable due to practical problems such as high model variability [5, 6], and out-of-distribution generalization issues [7, 8]. In settings that require making reliable and high-quality decisions, recent research has been working on designing the decision-making policy that can be adaptively in response to receiving low-quality predictions [4].

In this paper, we consider the setting where there are different sources of disturbance, each representing distinct (and often independent) resources or factors. For instance, in electricity grids, machine learning (ML) forecasts of variables like power consumption/generation fluctuations, electricity prices, and battery states can facilitate near-optimal operational decisions on the one hand, but on the other hand, latent variables such as power injections on certain loads can be highly unpredictable due to their volatility [9]. Another example is the drone navigation task [10], where the challenge

lies in managing external perturbations caused by a mixture of predictable elements like air flows and less predictable factors like raindrops. When performing online decision-making in this setting, one important technical challenge is to be able to combine disentanglement with an adaptive online learning algorithm to result in a more exact estimation of the trustfulness of the disturbance. To address this challenge, in this work, we focus on a linear quadratic control problem where system perturbations originate from unidentified and possibly heterogeneous latent resources/components.

**Contributions.** We develop a novel policy DISC that does the disentanglement and learns the adaptive parameter online simultaneously. We first introduce a policy, $\lambda$-CON, which extends the $\lambda$-confident approach in [4] by adapting a vectorized confidence parameter $\lambda \in [0,1]^k$, where each $\lambda(i)$ represents the estimated trustworthiness of the ML prediction for the $i$-th latent variable in the dynamical system. We then show that a static $\lambda$ cannot guarantee an optimal consistency and robustness tradeoff (see Definition 2.2 for a formal description) for $\lambda$-CON: if $\lambda$-CON is $(1 + o(1))$-*consistent*, then it is at least $\omega(1)$-*robust*.

To circumvent this limitation, we propose the dynamic policy DISC (Section 3.2). This algorithm leverages online learning to optimize the confidence parameter $\lambda_t$ at each time $t$. We establish competitive ratio guarantees for DISC in both linear and general mixing scenarios, as shown in Theorems 4.2 and 4.3 (Section 4) respectively. Under Assumption 1 and 2. The competitive ratio bound for DISC, outlined informally as follows, incorporates a term that embodies our "*best-of-both-worlds utilization*" of untrusted ML predictions:

$$\mathbb{E}\left[\mathsf{CR}(\mathrm{DISC})\right] \leq 1 + o(1) + O\left(\rho^{2w}\right) + \underbrace{O\left(\sum_{i=1}^{k} \frac{\overline{\varepsilon}(i)}{\Omega(T/w) + \overline{\varepsilon}(i)}\right)}_{\textit{Best-of-both-worlds utilization}}, \tag{1}$$

where the $o(1)$ term hides quantities that vanish when the total number of steps $T$ increases; $k$ is the number of latent variables generating the perturbations; $\rho \in (0,1)$; $w$ denotes the prediction window size and each $\overline{\varepsilon}(i)$ (for $i = 1, \ldots, k$) denotes the prediction error corresponding to each latent component. The last term in this result highlights the desired performance guarantee: When a component-wise prediction error $\overline{\varepsilon}(i)$ is small, the individual term $\overline{\varepsilon}(i)/\left(\Omega(T/w) + \overline{\varepsilon}(i)\right)$ will be negligible; otherwise, it always holds that $\overline{\varepsilon}(i)/\left(\Omega(T/w) + \overline{\varepsilon}(i)\right) \leq 1$, regardless of how high the prediction error becomes. Our proposed DISC is both $(1 + o(1))$-consistent and $O(1)$-robust with a sufficiently large prediction window size. We offer the first bound of this nature, grounded in a term as a function of the prediction error, without restrictive assumptions on the errors $(\overline{\varepsilon}(1), \ldots, \overline{\varepsilon}(k))$.

Proving our main result above is nontrivial due to the fact that despite it is known that an input-disturbed linear system can be reduced to an online convex optimization (OCO) with structured memory [1], the connection between the problem with $\lambda$-CON and a memoryless online optimization is not previously discovered. In Lemma 1, we provide a result that decouples the loss terms in the total regret in (9) and future perturbations, thereby reducing the problem of choosing $\lambda_t$ to an online optimization instance. Then we use a two-stage analysis as depicted in Figure 7 that combines a dynamic regret bound for our control policy and static regret bounds induced by online learning algorithms to derive the main result.

We demonstrate the practicality of DISC through two real-world examples (Section 5): a drone navigation problem with mixed external disturbances and voltage control in a power grid with heterogeneous power injections. We demonstrate that DISC, when applied with disentangled ML predictions, outperforms baselines that do not distinguish between underlying latent variables. Additionally, DISC shows remarkable adaptability to rapid changes in various scenarios, such as those involving ML models trained with out-of-distribution data in a non-stationary environment.

**Related Work.** Our work contributes to the growing community of algorithms with predictions while also incorporating ideas from adaptive control, online learning, and disentangled representation learning. We summarize the detailed related work in Appendix A.

## 2   Problem Formulation

*Notational conventions.* Throughout this paper, $\|\cdot\|$ denotes the $\ell_2$-norm for vectors and the matrix norm induced by the $\ell_2$-norm. We use a subscript $(\cdot)_t$ to represent a length-$k$ vector $s_t =$

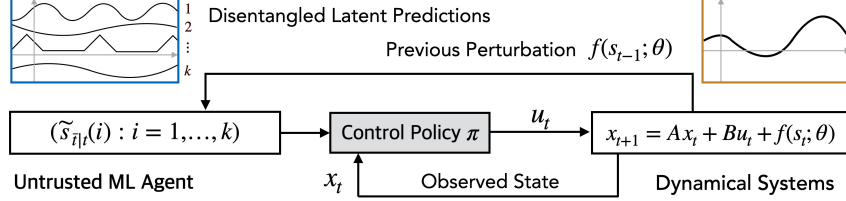

Figure 1: Overview of the system model considered in this work. **Left** time series: Disentangled latent variable time series predictions provided by an untrusted ML agent at time $t \in [T]$, represented by $(\widetilde{s}_{\bar{t}|t}(i) : i = 1, \ldots, k)$; **Right** time series: Observed mixed perturbations $(f(s_\tau; \theta) : \tau < t)$ at time $t - 1$.

$(s_t(1), \ldots, s_t(k))$ at time $t$ whose $i$-th coordinate is written as $s_t(i)$. We use $\lambda \circ s := (\lambda(i)s(i), i = 1, \ldots, k) \in \mathbb{R}^k$ to denote the Hadamard product between vectors $\lambda, s \in \mathbb{R}^k$.

## 2.1 Linear Quadratic Control with Latent Perturbations

We consider a finite-time linear dynamical system with *latent perturbations*. Let $[T] := \{0, \ldots, T-1\}$ be the set of time steps. Denote by $x_t \in \mathbb{R}^n$ a system state and $u_t \in \mathbb{R}^m$ an action from a state feedback policy $\pi_t$ at time $t \in [T]$. The system update rule is given by

$$x_{t+1} = Ax_t + Bu_t + f(s_t; \theta), \quad t \in [T] \tag{2}$$

where $s_t := (s_t(1), \ldots, s_t(k)) \in \mathbb{R}^k$ ($k \leq n$) contains $k$ latent variables (components) that together generate a perturbation at time $t \in [T]$ via a mixing function $f : \mathbb{R}^k \times \Theta \to \mathbb{R}^n$ parameterized by $\theta \in \Theta \subseteq \mathbb{R}^d$. The states and the actions are observed after being generated, while the mixing function $f$ transforms the unobservable latent variables in $s_t$ to an additive system perturbation. Throughout this paper, we focus on the nontrivial regime that $f(s_t; \theta) \neq \mathbf{0}_n$ for all $t \in [T]$.

In (2), $A \in \mathbb{R}^{n \times n}$ and $B \in \mathbb{R}^{n \times m}$ are system matrices. We consider the standard regime that the pair $(A, B)$ is stabilizable [4, 11]. Without loss of generality, we also assume the system is initialized with some fixed $x_0 \in \mathbb{R}^n$. The goal of control is to minimize the following quadratic costs given matrices $A, B, Q, R$:

$$J(\pi) := \sum_{t=0}^{T-1} \left( x_t^\top Q x_t + u_t^\top R u_t \right) + x_T^\top P x_T, \tag{3}$$

where $\pi := (\pi_t : t \in [T])$; $Q \in \mathbb{R}^{n \times n}, R \in \mathbb{R}^{m \times m}$ are positive definite matrices, and $P$ is a symmetric positive definite cost-to-go solution of the following discrete algebraic Riccati equation (DARE), which must exist because $(A, B)$ is stabilizable and $Q, R$ are positive definite [12]:

$$P = Q + A^\top P A - A^\top P B (R + B^\top P B)^{-1} B^\top P A.$$

Given $P$, we define $K := (R + B^\top P B)^{-1} B^\top P A$, and $u_t = -Kx_t$ is the feedback control law of a linear quadratic regulator (LQR), which is optimal in the case of zero disturbances ($w_t = 0$ for all $t \in [T]$). Further, let $F := A - BK$ be the closed-loop system matrix. The Gelfand's formula implies that there must exist a constant $C_F > 0$ and a spectral radius $\rho_F \in (0, 1)$ such that $\|F^t\| \leq C_F \rho_F^t$ for all $t \in [T]$.

In this work, we focus on designing a near-optimal policy $\pi = (\pi_t : t \in [T])$ that minimizes the total quadratic cost in (3) subject to the linear dynamics in (2). At time $t \in [T]$, each $\pi_t : \mathbb{R}^n \times \mathbb{R}^k \to \mathbb{R}^m$ is a state-feedback policy that produces an action $u_t \in \mathbb{R}^m$ with an observed state $x_{t-1}$, and ML predictions $(\widetilde{s}_{\bar{t}|t}(i) : i = 1, \ldots, k)$ of the latent variables $s_t \in \mathbb{R}^k$ for future $w$ time steps with a prediction window size $w > 0$ (denoting $\bar{t} := \min\{t + w - 1, T - 1\}$). Figure 1 above summarizes our system model. We defer a detailed introduction of the ML predictions in Section 2.2, together with the performance benchmarks considered in this paper. We relegate concrete latent perturbation modelling examples and real-world applications to Appendix B.

In light of the existing nonlinear ICA models summarized in Appendix B, throughout this paper, we assume the continuity and invertibility conditions hold for the mixing function $f$ to guarantee

our theoretical results. This assumption is weaker compared to those required in the nonlinear ICA literature [13, 14, 15, 16, 17] to guarantee identifiability, as summarized in Table 1 (see Appendix B), noting that our goal is instead to provide a near-optimal competitive ratio bound regardless of the error of disentanglement and latent variable predictions (see our performance benchmarks defined in Section 2.2).

**Assumption 1.** The mixing function $f : \mathbb{R}^k \times \Theta \to \mathbb{R}^n$ is Lipschitz continuous and bijective with respect to $s \in \mathbb{R}^k$.

## 2.2 Performance Benchmark

Let $\widetilde{s}_{\tau|t}$ be the latent variable time series value at time $\tau$, predicted by an untrusted ML agent at time $t \leq \tau$. The prediction window size is an integer $w > 0$. We write $\bar{t} := \min\{t + w - 1, T - 1\}$. The estimated mixing parameter at each time $t \in [T]$ is denoted by $\widetilde{\theta}_t$. Define the following error terms[2] for all $t \leq \tau \leq \bar{t}$ and $t \in [T]$:

$$\big(\varepsilon_{\tau|t}(1), \ldots, \varepsilon_{\tau|t}(k)\big) = \varepsilon_{\tau|t} := s_\tau - \widetilde{s}_{\tau|t}, \quad \eta_t := \widetilde{\theta}_t - \theta. \tag{4}$$

To assess the cumulative impact of prediction error over all $T$ time steps, we define the total prediction error corresponding to the component-wise latent variable time-series prediction and the mixing parameter estimation as follows:

$$\bar{\varepsilon}(i) := \sum_{t=0}^{T-1} \sum_{\tau=t}^{\bar{T}} \big(\rho_F^{\tau-t} \varepsilon_{\tau|t}(i)\big)^2, \quad \bar{\eta} := \sum_{t=0}^{T-1} \|\eta_t\|^2, \tag{5}$$

where $\rho_F \in (0, 1)$ represents the spectral radius of the matrix $F$. The term $\rho_F^{\tau-t}$ in (5) accounts for the exponential decay phenomenon in the impact of component-wise errors, implying that predictions further into the future contribute less to the total error. This approach to error measurement is analogous to methods employed in linear quadratic control models with adaptive offline adversarial perturbations, as discussed in [4], which provides a foundational understanding of error evaluation in our context.

Our performance benchmark is the competitive ratio for a given prediction error $\varepsilon$, defined as follows. To be more precise, we focus on the competitive ratio defined for deterministic online algorithms with an adaptive offline adversary that selects the system parameters $A, B, Q, R, f, \theta$ and latent time series $(s_t : t \in [T])$. Write $\varepsilon := (\bar{\varepsilon}(i) : i = 1, \ldots, k)$ and denote by $J(\pi; \varepsilon)$ the total cost obtained by implementing $\pi = (\pi_t : t \in [T])$ with some fixed prediction error $\varepsilon$.

**Definition 2.1.** Fix some prediction error $\varepsilon$. The *competitive ratio* $\mathsf{CR}(\pi; \varepsilon)$ is defined as the smallest constant $C \geq 1$ such that $J(\pi; \varepsilon) \leq C \cdot J^\star$ for all $A, B, Q, R, f, \theta$, and $(s_t : t \in [T])$ satisfying the model assumptions.

The following notions of consistency and robustness with respect to the competitive ratio offer a concise characterization to measure the algorithmic performance in the presence of prediction error $\varepsilon$. It aligns with the benchmarks used in the growing literature on algorithms with predictions [18, 19, 20, 21] (see further discussions provided in Section A).

**Definition 2.2.** A policy $\pi$ is $\gamma$-*consistent* if its competitive ratio satisfies $\mathsf{CR}(\pi; \varepsilon) \leq \gamma$ for $\varepsilon = 0$ and $\kappa$-*robust* if $\mathsf{CR}(\pi; \varepsilon) \leq \kappa$ for all $\varepsilon$.

# 3 Disentangled Confident Policy

In this section, we present our policy, DISC, which takes disentangled and untrusted ML predictions and achieves near-optimal consistency and robustness tradeoff (see Definition 2.2).

**Algorithm 1:** DISentangled Confidence (DISC) policy

---

**for** $t = 0, \ldots, T-1$ **do**

    $\bar{t} \leftarrow \min\{t + w - 1, T - 1\}$

    /* Online learning of $\lambda_t$                                                          */

    **if** $t = 0$ **then** Initialize $\lambda_0 \in \mathcal{I}$

    **else** Set $\lambda_t = $ ONLINE-PROCEDURE $(\zeta_\ell : \ell \in [t])$ through (12) or (13) with $\zeta_\ell$ defined in (11)

    /* Obtain disentangled predictions                                          */

    Get $\left(\widetilde{s}_{\tau|t} : t \le \tau \le \bar{t}\right)$ and an estimated mixing parameter $\widetilde{\theta}_t$

    /* Implement disentangled $\lambda_t$-confident policy                        */

    Take $u_t$ as in (6) with a learned $\lambda_t$ such that

$$u_t = -Kx_t - Y\sum_{\tau=t}^{\bar{t}} \left(F^\top\right)^{\tau-t} Pf\left(\lambda_t \circ \widetilde{s}_{\tau|t}; \widetilde{\theta}_t\right), \quad (\lambda_t\text{-CON POLICY})$$

    Update and observe $x_{t+1}$ following (2)

**end**

---

## 3.1 Warm-Up: Disentangled $\lambda$-Confident Policy ($\lambda$-CON)

Before proceeding to introduce DISC, we first consider a policy that balances consistency and robustness by combining an MPC policy that fully utilizes the untrusted ML predictions, and an LQR without considering any predictions.

Denote by $\bar{t} := \min\{t + w - 1, T - 1\}$ and let $Y := (R + B^\top PB)^{-1}B^\top$. Recall that as defined in Section 2.2, at each time $t \in [T]$, $\left(\widetilde{s}_{\tau|t} : t \le \tau \le \bar{t}\right)$ represents a sequence of predictions for the future values of the latent variables $(s_\tau : t \le \tau \le \bar{t})$, with these predictions spanning a predetermined window size $w > 0$. Similarly, the sequence $(\theta_t : t \in [T])$ corresponds to the estimated mixing parameters. At each time $t \in [T]$, an estimate $\theta_t$ is obtained by applying some disentanglement algorithm (see those examples in Section B). With this setup, we proceed to define an MPC-type action as follows:

$$u_t = -Kx_t - Y\sum_{\tau=t}^{\bar{t}} \left(F^\top\right)^{\tau-t} Pf\left(\lambda \circ \widetilde{s}_{\tau|t}; \widetilde{\theta}_t\right), \quad (\lambda\text{-CON POLICY}) \qquad (6)$$

which specifies a *disentangled $\lambda$-confident policy*, denoted by $\lambda$-CON, where $\lambda \in \mathcal{I} := [0,1]^k$ is a fixed *trust parameter*. The term $\lambda \circ \widetilde{s}_{\tau|t}$ denotes the Hadamard product of $\lambda$ and $\widetilde{s}_{\tau|t}$. It is worth noting that it is well known that (6) is an optimal solution of the following MPC scheme [4, 2]:

$$\min_{u \in \mathbb{R}^m} \sum_{\tau=t}^{\bar{t}} \left(x_\tau^\top Qx_\tau + u_\tau^\top Ru_\tau\right) + x_{\bar{t}+1}^\top Px_{\bar{t}+1} \qquad (7)$$

$$\text{s.t. } x_{\tau+1} = Ax_\tau + Bu_\tau + f\left(\lambda \circ \widetilde{s}_{\tau|t}; \widetilde{\theta}_t\right), \tau \in \left[t, \bar{t}\right]. \qquad (8)$$

In particular, if $\lambda$ is an all-one vector $\mathbf{1}_k$ in (7), it trusts the ML predictions. Note that it generalizes the $\lambda$-confident policy in [4], which linearly combines actions from an MPC policy and an LQR. In Section 4.1, we present a negative result for $\lambda$-CON, indicating that it cannot achieve the optimal consistency and robustness tradeoff (see Definition 2.2). This motivates the disentangled confident policy discussed in the next section.

## 3.2 Disentangled Confidence Policy (DISC)

At each time $t \in [T]$, the algorithm adaptively learns a *trust parameter* $\lambda_t \in \mathcal{I} := [0,1]^k$, and uses a Hadamard product of $\lambda_t$ and the ML learned latent variable $\lambda_t \circ \widetilde{s}_{\tau|t}$ to estimate future perturbations. The update rule of the trust parameter $\lambda_t$ follows an ONLINE-PROCEDURE, which can be constructed by the following explicit form of the overall dynamic regret (see [22, 4]) with a fixed $\lambda \in \mathcal{I}$:

$$J(\pi(\lambda)) - J^\star = \sum_{\ell=0}^{T-1} \psi_{\ell,T}^\top(\lambda)H\psi_{\ell,T}(\lambda), \qquad (9)$$

where $H := B(R + B^\top P B)^{-1} B^\top$, and $\psi_{\ell,T}(\lambda)$ is defined as

$$\psi_{\ell,T}(\lambda) := \sum_{\tau=\ell}^{\min\{\ell+w-1,T-1\}} \left(F^\top\right)^{\tau-\ell} P \left( f\left(s_\tau; \theta_\tau\right) - f\left(\lambda \circ \widetilde{s}_{\tau|\ell}; \widetilde{\theta}_\ell\right) \right). \tag{10}$$

Consider the application of online optimization strategies to minimize total regret as depicted in (9). At each time $t$, we obtain $\psi_{\ell,0}(\lambda), \ldots, \psi_{\ell,t}(\lambda)$. For $\psi_{\ell,t}(\lambda)$, since the summation is over $\tau \in \{\ell, \min\{\ell + w - 1, t - 1\}\}$, $\psi_{\ell,t} : \mathcal{I} \to \mathbb{R}^n$ is a function that depends on $t \in [T]$. Thus, the formulation of the offline optimization (9) does not conform to a canonical framework suitable for online optimization. This necessitates a tailored approach for online learning, which is facilitated by Lemma 1, allowing for the online learning of $\lambda$. Below we assume the following to regulate the learned sequence $(\lambda_t : t \in [T])$, which holds for typical online learning algorithms with stationary environments [23, 24, 25]. Note that when $t - \tau < 0$, we consider $\lambda_{t-\tau} = \lambda_0$.

**Assumption 2.** If $\tau > 0$ is a constant, then $(\lambda_t : t \in [T])$ satisfies that $\sum_{t=0}^{T-1} |\lambda_t - \lambda_{t-\tau}| = o(T)$.

Define $\underline{\ell} := \max\{\ell - w + 1, 0\}$. The following lemma helps convert (9) to a form that can be reduced to an online optimization problem. The proof can be found in Appendix D.3.

**Lemma 1.** *Define a function $\zeta_\ell : \mathcal{I} \to \mathbb{R}^n$ as $\zeta_\ell(\lambda) := \sum_{\tau=\underline{\ell}}^{\ell} g^\top(\tau, \ell) H g(\tau, \ell)$, where*

$$g(\tau, \ell) := \left(F^\top\right)^{\tau-\ell} P \left( f\left(s_\tau; \theta_\tau\right) - f\left(\lambda \circ \widetilde{s}_{\tau|\ell}; \widetilde{\theta}_\ell\right) \right). \tag{11}$$

*Then for any $t \in [T]$, it follows that $\sum_{\ell=0}^{t-1} \psi_{\ell,t}^\top(\lambda) H \psi_{\ell,t}(\lambda) \leq w \sum_{\ell=0}^{t-1} \zeta_\ell$.*

Note that each $\zeta_\ell(\lambda)$ defines a convex function of $\lambda$, as verified in Appendix D.1. Therefore, the selection of $\lambda_t$ at each time $t \in [T]$ follows from an online learning procedure ONLINE-PROCEDURE, whose concrete realizations include the follow-the-regularized-leader (FTRL) approach with an $\ell_2$-norm regularizer [23, 24], which is equivalent to online mirror descent (OMD) when $f$ is convex; and the follow-the-perturbed-leader (FTPL) [26, 27, 28, 25] for online non-convex learning, taken previous quantities $(\zeta_\ell : \ell \in [t])$ as the input. Next, we will discuss theoretical guarantees for linear and general mixing models with specified online learning procedures, as detailed later in Section 4. Our policy is summarized in Algorithm 1.

# 4 Main Results

In this section, we present our technical results, proving that the scheme in DISC achieves near-optimal competitive ratio bounds for both linear and general mixing cases. Proving our main result above is nontrivial due to the fact that despite it is known that an input-disturbed linear system can be reduced to an online convex optimization (OCO) with structured memory [1], the connection between the problem with $\lambda$-CON and a memoryless online optimization is not previously discovered. In Lemma 1, we provide a result that decouples the dependency between cost functions in our problem that depends on previous actions via a linear dynamical system (see the dynamical system defined in (2)), thereby reducing the problem of choosing $\lambda_t$ to an online optimization instance, then use a two-stage analysis (see Figure 7) that combines the dynamic regret analysis of $\lambda$-CON and static regret bounds corresponding to applying online optimization algorithms to learn a confidence parameter.

## 4.1 A Fundamental Gap

First, we motivate the necessity of learning $(\lambda_t : t \in [T])$ online. Based on Definition 2.2, the following theorem reveals a negative result of $\lambda$-CON, since for any fixed non-zero $\lambda$, there always exists predicted time series $\left(\widetilde{s}_{\tau|t} : t \leq \tau \leq \bar{t}, t \in [T]\right)$ such that the competitive ratio $\mathsf{CR}(\lambda\text{-CON})$ for $\lambda$-CON can be unbounded (when $T$ goes to infinity).

**Theorem 4.1** (Consistency-Robustness Impossibility for $\mathsf{CR}(\lambda\text{-CON})$)**.** *If the $\lambda$-confident policy $\lambda$-CON is $(1 + o(1))$-consistent, then it is at least $\omega(1)$-robust, even if the mixing parameter estimate is perfect, i.e., $\overline{\eta} = 0$.*

Here, $o(\cdot)$ and $\omega(\cdot)$ characterize the asymptotic asymptotic growth rates with respect to the time horizon length $T$. Next, we show in both the linear and general mixing settings, CR(DISC) can be bounded as previewed in Section 1, thus the gap between CR($\lambda$-CON) and CR(DISC) can be arbitrarily large as implied by Theorem 4.1 above. Especially, Corollary B.1 in Section 4.2 implies that DISC is both $(1 + o(1))$-consistent and $O(1)$-robust, with a sufficiently large prediction window size $w = \omega(1)$, revealing that online learning of the confidence parameter provides a significant improvement of the consistency and robustness tradeoff.

## 4.2   Linear Mixing

We consider a linear mixing setting, where the mixing function $f$ is linear such that $f(s; \theta) = \theta s$ for all $s \in \mathbb{R}^k$ where $\theta = (\theta_{ij})$ denotes an $n \times k$ full column rank mixing matrix. Suppose $(s_t : t \in [T])$ and $\theta$ are bounded such that $\|s_t\| \leq \overline{s}$ for all $t \in [T]$ and $\|\theta\| \leq \overline{\theta}$. For this particular instance, we implement a (follow-the-regularized-leader) FTRL procedure that sets

$$\lambda_t \in \underset{\lambda \in \mathcal{I}}{\arg\min} \left( \sum_{\ell=0}^{t-1} \nabla_\lambda^\top \left( \zeta_\ell(\lambda) \right) \lambda + \frac{1}{\beta} \|\lambda - \lambda_0\|^2 \right) \quad \text{(FTRL FOR } \lambda\text{-LEARNING)} \qquad (12)$$

at each time $t \in [T]$ for some $\beta > 0$ that can be optimized. The DISC policy satisfies the following competitive ratio bound, whose proof can be found in Appendix E.

**Theorem 4.2** (DISC for Linear Mixing). *With our model assumptions, the competitive ratio of* DISC *with a linear mixing function satisfies*

$$\mathrm{CR(DISC)} \leq 1 + O \left( \sum_{i=1}^{k} \frac{\overline{\varepsilon}(i)}{\Omega(T/w) + \overline{\varepsilon}(i)} \right) + O \left( w \sqrt{\frac{k}{T}} \right) + O \left( \rho_F^{2w} + \frac{\overline{\eta}}{T} \right),$$

*where $k$ is the number of latent variables; $T$ is the time horizon length; $\rho_F \in (0, 1)$ is the spectral radius of $F$; $(\overline{\varepsilon}(i) : i = 1, \ldots, k)$ and $\overline{\eta}$ are defined in (5); $O(\cdot)$ and $\Omega(\cdot)$ hide multiplicative constants (see the details in Appendix E).*

This result highlights a consistency and robustness tradeoff of DISC, as claimed in Section 1, which offers a dual advantage: it exploits accurate predictions when available, and safeguards against the ramifications of trusting inaccurate forecasts, thus ensuring system performance reliability. Besides, our analysis can be carried out to convex mixing functions, as the online learning of $\lambda_t$ in (12) via FTRL guarantees a sub-linear static regret as long as $\zeta_\ell(\lambda)$ is convex in $\lambda$ for all $\ell \in [T]$. We further provide a bound for sample efficient ICA in the appendix.

## 4.3   General Mixing

In this section, we extend the linear assumption on $f$ by considering a general setting where the mixing function $f$ is an arbitrary Lipschitz continuous function (see Assumption 1). To deal with the non-convexity of $f$, we revisit the (follow-the-perturbed-leader) FTPL approach (see [28, 25]) and consider the following online learning procedure for tuning $(\lambda_t : t \in [T])$:

$$\lambda_t \in \underset{\lambda \in \mathcal{I}}{\arg\min} \left( \sum_{\ell=0}^{t-1} \zeta_\ell(\lambda) + \sigma_t^\top \lambda \right) \quad \text{(FTPL FOR } \lambda\text{-LEARNING)}. \qquad (13)$$

Here, $\sigma_t$ is a length-$k$ random vector with each coordinate $\sigma_t(i)$ $(i = 1, \ldots, k)$ being an IID random variable from an exponential distribution. The DISC policy satisfies the following competitive ratio bound.

**Theorem 4.3** (DISC for General Mixing). *With our model assumptions, the expected competitive ratio of* DISC *satisfies*

$$\mathbb{E}\left[\mathrm{CR(DISC)}\right] \leq 1 + O \left( \sum_{i=1}^{k} \frac{\overline{\varepsilon}(i)}{\Omega(T/w) + \overline{\varepsilon}(i)} \right) + O \left( \overline{s} \rho_F^{2w} + \frac{\overline{\eta}}{T} \right) + O \left( \frac{wk^2}{\sqrt{T}} \right), \qquad (14)$$

*where the parameters $\rho_F, w, k, T, \overline{\eta}$, and $(\overline{\varepsilon}(i) : i = 1, \ldots, k)$ are the same as in Theorem 4.2 and $\|s_t\| \leq \overline{s}$ for all $t \in [T]$. The expectation is taken over the randomly sampled $(\sigma_t : t \in [T])$.*

The notation $O(\cdot)$ and $\Omega(\cdot)$ above in Theorem 4.3 hide multiplicative constants such as the Lipschitz constant of the mixing function $f$ (details are provided in Appendix H). The result above implies the best-of-both-worlds prediction utilization as previewed in Section 1, except that the online learning of $\lambda_t$ converges slower with a rate $k^2$, causing the term $O(k^2/\sqrt{T})$ in the competitive ratio bound (14). Unlike the linear setting, to ensure a near-optimal expected competitive ratio, the latent time series $(s_t : t \in [T])$ is not necessarily bounded, as long as the prediction window size $w$ is sufficiently large (e.g., $w = \Theta(\log T)$) so that the term $\overline{s}\rho_F^{2w}$ vanishes. Finally, note that the online learning procedure of $\lambda_t$ is not limited to FTPL. For example, the non-convex online optimization algorithm in [29] can be used to optimize $(\lambda_t : t \in [T])$ online and leads to a similar guarantee. The proof of Theorem 4.3 can be found in Appendix H.

## 5 Experiments

**Experimental Setup.** To generate ML predictions, we use the FastICA method in [30] to decompose the mixed perturbations and train a multi-layer perceptron neural network with 4 hidden layers to predict the future latent variables. FastICA is a simple and efficient linear ICA method that aims to find an orthogonal rotation of the observation that maximizes the rotated components' non-Gaussianity using fixed-point iteration. Furthermore, in our experiments, we implement a follow-the-regularized-leader (FTRL) optimization with a $\ell_2$-regularizer $\|\lambda - \lambda_0\|^2$, which is equivalent to the following equivalent online mirror descent (OMD) implementation [24]: $\tau_t = \tau_{t-1} - \frac{\beta}{2}\nabla_\lambda\left(\zeta_t^\top H \zeta_t\right), \ \lambda_t = \mathsf{Proj}_{\mathcal{T}}(\tau_t)$. The detailed parameters and hyper-parameters used in our experiments can be found in Appendix C.3. The code is available at https://github.com/tspbfs/DisentangleControl.

### 5.1 Experiment A: Drone Navigation with Mixed Disturbances

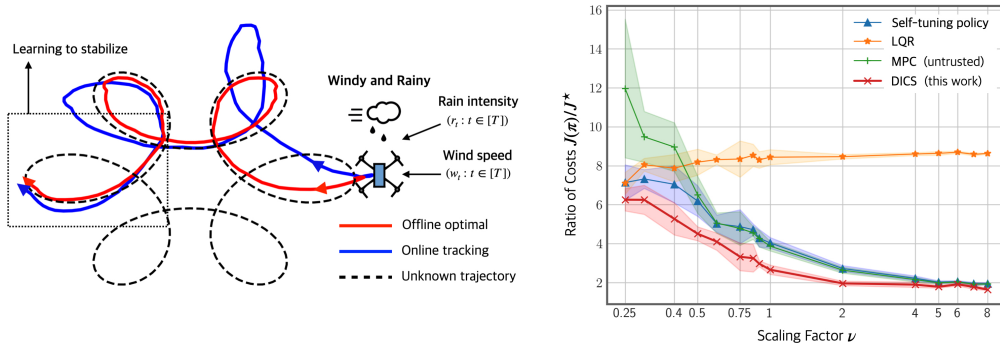

Figure 2: **Drone Navigation** under challenging windy and rainy weather conditions, with the drone's target path illustrated by a dotted curve. The external perturbations impacting the drone's flight are modeled by two independent, time-varying latent variables: $w_t$ for wind speed and $r_t$ for rain intensity at time $t \in [T]$. The trajectory produced by the DISC policy (Algorithm 1 in Section 3) is represented by the blue curve, while that from the offline optimal policy is traced in red. More comparison results with other baseline policies can be found in Appendix C.2. Right: **Ratio of Costs** $J(\pi)/J^\star$ ($y$-axis) between DISC (red), linear quadratic regulator (LQR, orange), model predictive control with untrusted ML predictions (MPC (UNTRUSTED), green), and the self-tuning policy (blue), with a varying scaling factor $\nu$ from 0.25 to 8. Shadow area depicts the range of standard deviations for 5 random tests.

We first consider the drone piloting task described in Example 1 (a detailed description is delegated to Appendix B) to track an unknown trajectory, shown on the left of Figure 2.

**Competitive Ratio Analysis.** We present a comparative analysis of the ratio of costs achieved by our DISC policy (detailed in Algorithm 1 in Section 3) on the right of Figure 2, against several benchmarks: the Linear Quadratic Regulator (LQR), Model Predictive Control (MPC) using untrusted ML predictions (i.e., setting $\lambda = \mathbf{1}_k$ in Equation (7)), and the self-tuning policy outlined in [4] and [31].

We examine the impact of varying a scaling factor $\nu$ in the range 0.25 to 8, adjusting the additive perturbation $r_t c_2$ in (16) to $(r_t/\nu)c_2$. This scaling factor $\nu$ serves as a proxy for different rain intensities, influencing the level of environmental unpredictability—the larger $\nu$ is, the more predictable the

environment. With the theoretically guaranteed best-of-both-worlds utilization of disentangled and untrusted ML predictions, DISC consistently outperforms existing baselines in terms of cost ratios across varying levels of environmental predictability.

## 5.2 Example B: Voltage Control with Heterogeneous Power Injections

We consider the voltage control task outlined in Example 2 for a distribution power grid with 11 nodes, as depicted in Figure 3. Specifically, nodes 3, 5, and 8 are active, each supplying dynamic active power injections $(p_t(i) : i = 3, 5, 8)$ at every time step $t \in [T]$. Collectively, these nodes induce the composite perturbation $Gp_t + v_0 \mathbf{1}_n$. The objective is to apply the DISC algorithm to determine near-optimal controllable reactive power injections that will effectively mitigate voltage fluctuations across the grid. More details of the problem setting can be found in Appendix C.4.

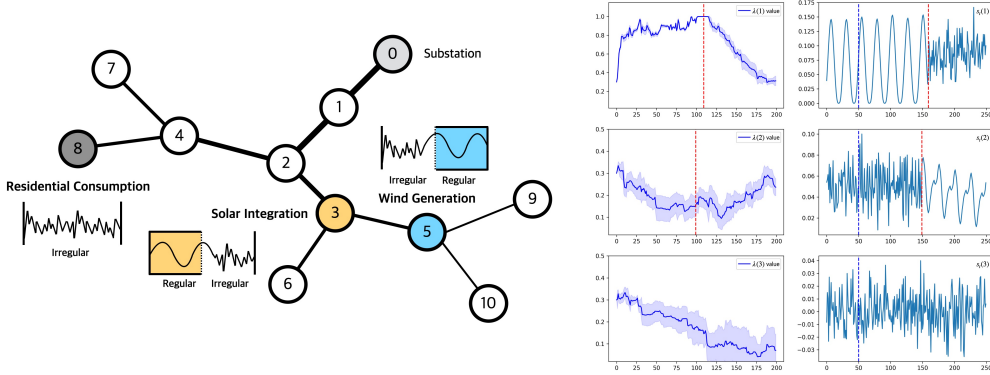

Figure 3: **Voltage Control** with in a dynamic environment. This figure illustrates the heterogeneity and variability of power injections at different nodes within an electrical grid, including a residential area, a solar photovoltaic (PV) system, and a wind turbine. Right: **Example B.** *Voltage Control (Section 5.2).* **Left column**: *Convergence of confidence parameters* $\lambda(1), \lambda(2), \lambda(3)$ *corresponding to* 3 *latent components.* **Right column**: *Temporal dynamics of latent time series* $s_t$ *in* $\mathbb{R}$. We illustrate the time series data for three key latent variables in the energy system (see Figure 3): Gaussian-distributed residential consumption (bottom), real-world photovoltaic (PV) integration (top), and wind generation (middle). The red dotted line marks a critical transition point where there is a notable shift in the power generation patterns. We use time series before the blue dotted line as the buffered data to warm start the ML model, learn the mixing matrix, and obtain disentangled predictions. The $x$-axis in each graph marks the time steps, while the $y$-axis denotes the values of the time series.

We consider a realistic scenario when and use this to demonstrate the adaptivity of DISC in changing environments. Figure 3 shows the change of patterns corresponding to the solar integration and wind generation. We set $T = 200$. At time $t = 110$, the solar generation switches from a regular pattern to generating random Gaussian noise, representing miscommunication or system faults. Similarly, at time $t = 100$, the wind generation is recovered from the irregular mode to a regular mode. For the regular solar and wind generation time series, we use the real solar PV generation time series data from the DTU-Data in 2021 [32] and wind generation from the U.S. Virgin Islands Wind Resources from the National Renewable Energy Lab (NREL) [33].

**Adaptability Amidst Environmental Variability.** Figure 3 exhibits the adaptability of our approach in response to environmental shifts in renewable energy generation and consumption patterns. Specifically, the figure presents the time series for solar generation $(s_t(1) : t \in [T])$, wind generation $(s_t(2) : t \in [T])$, and residential power consumption $(s_t(3) : t \in [T])$. The transition points in the time series patterns are marked by dotted red lines in the figure.

A notable observation is the reactive behavior of the confidence parameters to the changes in predictability within each time series. For instance, as the solar generation transitions to an irregular Gaussian noise profile at $t \geq 110$, this unpredictability precipitates a decline in the corresponding confidence parameter $\lambda_t(1)$. Conversely, the wind generation time series exhibits a move towards a more predictable pattern past $t \geq 100$, resulting in a progressive increase in the confidence parameter $\lambda_t(2)$. In the case of residential consumption, which is consistently modeled as Gaussian

noise, the algorithm adaptively learns to reduce the confidence parameter $\lambda(3)$. The resilience of the DISC algorithm in dynamically adjusting confidence levels demonstrates its robustness against environmental shifts, a critical feature for real-world applications where conditions are subject to sudden and unpredictable changes. This further highlights the efficacy of DISC in maintaining system stability and performance through intelligent adaptability to the reliability of input data, as verified by the theoretical results in Section 4.

## 6 Concluding Remarks

Our work combines online control and the concept of learning disentangled predictions. In this paper, we have introduced a novel policy DISC that achieves the best-of-both-world utilization of untrusted ML predictions of latent variables for a linear control problem, where the advantage of disentanglement is theoretically validated. The practicality of our method has been validated through two real-world applications, which exemplify its relevance and adaptability to complex, real-world scenarios.

**Limitations and Future Directions.** Despite that our main results, i.e., the competitive ratio bounds in Theorem 4.2 and 4.3 do not rely on the assumption that the latent variables $(s_t(1), \ldots, s_t(k))$ are independent, most of the disentanglement methods do (like sample efficient ICA algorithms used to derive Corollary B.1 and the detailed assumptions are summarized in Table 1). It would be interesting to explore more carefully designed algorithms to guarantee both sample efficiency and identifiability based on our control model. Looking ahead, our results open several intriguing paths. An immediate extension of particular interest is adapting our methodology to nonlinear dynamical systems, which could strengthen our current results. Furthermore, our work contributes to the growing community of algorithms with predictions, a compelling question arises: can the best-of-both-worlds competitive ratio bounds we have achieved be replicated across different online decision-making problems? Obtaining either positive or negative results would potentially lead to broad implications for the field of robust control and online learning.

## Acknowledgement

We would like to thank all the anonymous reviewers for their helpful comments.

Tongxin Li was supported in part by the National Natural Science Foundation of China (NSFC) under grant No.72301234, the GuangDong Basic and Applied Basic Research Foundation, NSFC under grant No. 62336005, the PengCheng Peacock Supporting Scientific Research Fund Category C (2024TC0024), the GuangDong Provincial Key Laboratory of Mathematical Foundations for Artificial Intelligence (2023B1212010001), and the Shenzhen Key Lab of Crowd Intelligence Empowered Low-Carbon Energy Network (No. ZDSYS20220606100601002).

## Footnotes

*Correspondence to: Tongxin Li <`litongxin@cuhk.edu.cn`>. Tongxin Li contributed to the design of the algorithms and experiments, and the writing of the paper. Hao Liu focused on conducting the experiments and data analysis. The contributions are equal.

[2]We define two pairs $(s, \theta)$ and $(s', \theta')$ as *consistent* if they yield identical outputs in the mixing function $f$, i.e., $f(s; \theta) = f(s'; \theta')$. For simplicity in our notation, we will treat consistent pairs $(s, \theta)$ and $(s', \theta')$ as indistinguishable when they produce the same perturbation effect in the system. Here, at each time step $t \in [T]$, the estimated mixing parameter and the time series predictions are compared against the consistent pairs $\theta$ and $s_\tau$ with the smallest prediction error.

## References

[1] Guanya Shi, Yiheng Lin, Soon-Jo Chung, Yisong Yue, and Adam Wierman. Online optimization with memory and competitive control. *Advances in Neural Information Processing Systems*, 33:20636–20647, 2020.

[2] Chenkai Yu, Guanya Shi, Soon-Jo Chung, Yisong Yue, and Adam Wierman. The power of predictions in online control. *Advances in Neural Information Processing Systems*, 33, 2020.

[3] Tongxin Li, Yue Chen, Bo Sun, Adam Wierman, and Steven Low. Information aggregation for constrained online control. *ACM SIGMETRICS Performance Evaluation Review*, 49(1):7–8, 2021.

[4] Tongxin Li, Ruixiao Yang, Guannan Qu, Guanya Shi, Chenkai Yu, Adam Wierman, and Steven Low. Robustness and consistency in linear quadratic control with untrusted predictions. *ACM SIGMETRICS Performance Evaluation Review*, 50(1):107–108, 2022.

[5] Tobias Johannink, Shikhar Bahl, Ashvin Nair, Jianlan Luo, Avinash Kumar, Matthias Loskyll, Juan Aparicio Ojea, Eugen Solowjow, and Sergey Levine. Residual reinforcement learning for robot control. In *2019 International Conference on Robotics and Automation (ICRA)*, pages 6023–6029. IEEE, 2019.

[6] Benjamin Recht. A tour of reinforcement learning: The view from continuous control. *Annual Review of Control, Robotics, and Autonomous Systems*, 2:253–279, 2019.

[7] Chaochao Lu, Yuhuai Wu, José Miguel Hernández-Lobato, and Bernhard Schölkopf. Invariant causal representation learning for out-of-distribution generalization. In *International Conference on Learning Representations*, 2021.

[8] Florian Wenzel, Andrea Dittadi, Peter Gehler, Carl-Johann Simon-Gabriel, Max Horn, Dominik Zietlow, David Kernert, Chris Russell, Thomas Brox, Bernt Schiele, et al. Assaying out-of-distribution generalization in transfer learning. *Advances in Neural Information Processing Systems*, 35:7181–7198, 2022.

[9] Evan D Sherwin, Max Henrion, and Inês ML Azevedo. Estimation of the year-on-year volatility and the unpredictability of the united states energy system. *Nature Energy*, 3(4):341–346, 2018.

[10] Tiago Oliveira, A Pedro Aguiar, and Pedro Encarnacao. Moving path following for unmanned aerial vehicles with applications to single and multiple target tracking problems. *IEEE Transactions on Robotics*, 32(5):1062–1078, 2016.

[11] Sungho Shin, Yiheng Lin, Guannan Qu, Adam Wierman, and Mihai Anitescu. Near-optimal distributed linear-quadratic regulator for networked systems. *SIAM Journal on Control and Optimization*, 61(3):1113–1135, 2023.

[12] Geir E Dullerud and Fernando Paganini. *A course in robust control theory: a convex approach*, volume 36. Springer Science & Business Media, 2013.

[13] Aapo Hyvarinen and Hiroshi Morioka. Unsupervised feature extraction by time-contrastive learning and nonlinear ica. *Advances in neural information processing systems*, 29, 2016.

[14] Aapo Hyvarinen and Hiroshi Morioka. Nonlinear ica of temporally dependent stationary sources. In *Artificial Intelligence and Statistics*, pages 460–469. PMLR, 2017.

[15] Ilyes Khemakhem, Diederik Kingma, Ricardo Monti, and Aapo Hyvarinen. Variational autoencoders and nonlinear ica: A unifying framework. In *International Conference on Artificial Intelligence and Statistics*, pages 2207–2217. PMLR, 2020.

[16] Xiaojiang Yang, Yi Wang, Jiacheng Sun, Xing Zhang, Shifeng Zhang, Zhenguo Li, and Junchi Yan. Nonlinear ica using volume-preserving transformations. In *International Conference on Learning Representations*, 2021.

[17] Yujia Zheng, Ignavier Ng, and Kun Zhang. On the identifiability of nonlinear ica: Sparsity and beyond. *Advances in Neural Information Processing Systems*, 35:16411–16422, 2022.

[18] Manish Purohit, Zoya Svitkina, and Ravi Kumar. Improving online algorithms via ml predictions. In *Advances in Neural Information Processing Systems*, pages 9661–9670, 2018.

[19] Alexander Wei and Fred Zhang. Optimal robustness-consistency trade-offs for learning-augmented online algorithms. *arXiv preprint arXiv:2010.11443*, 2020.

[20] Shom Banerjee. Improving online rent-or-buy algorithms with sequential decision making and ml predictions. *Advances in Neural Information Processing Systems*, 33:21072–21080, 2020.

[21] Nicolas Christianson, Tinashe Handina, and Adam Wierman. Chasing convex bodies and functions with black-box advice. In *Conference on Learning Theory*, pages 867–908. PMLR, 2022.

[22] Chenkai Yu, Guanya Shi, Soon-Jo Chung, Yisong Yue, and Adam Wierman. Competitive control with delayed imperfect information. In *2022 American Control Conference (ACC)*, pages 2604–2610. IEEE, 2022.

[23] Adam Kalai and Santosh Vempala. Efficient algorithms for online decision problems. *Journal of Computer and System Sciences*, 71(3):291–307, 2005.

[24] Elad Hazan and Satyen Kale. Extracting certainty from uncertainty: Regret bounded by variation in costs. *Machine learning*, 80:165–188, 2010.

[25] Arun Sai Suggala and Praneeth Netrapalli. Online non-convex learning: Following the perturbed leader is optimal. In *Algorithmic Learning Theory*, pages 845–861. PMLR, 2020.

[26] James Hannan. Approximation to bayes risk in repeated play. *Contributions to the Theory of Games*, 3:97–139, 1957.

[27] Walid Krichene, Maximilian Balandat, Claire Tomlin, and Alexandre Bayen. The hedge algorithm on a continuum. In *International Conference on Machine Learning*, pages 824–832. PMLR, 2015.

[28] Naman Agarwal, Alon Gonen, and Elad Hazan. Learning in non-convex games with an optimization oracle. In *Conference on Learning Theory*, pages 18–29. PMLR, 2019.

[29] Lin Yang, Lei Deng, Mohammad H Hajiesmaili, Cheng Tan, and Wing Shing Wong. An optimal algorithm for online non-convex learning. *Proceedings of the ACM on Measurement and Analysis of Computing Systems*, 2(2):1–25, 2018.

[30] Aapo Hyvärinen and Erkki Oja. Independent component analysis: algorithms and applications. *Neural networks*, 13(4-5):411–430, 2000.

[31] Yiheng Lin, James A Preiss, Emile Timothy Anand, Yingying Li, Yisong Yue, and Adam Wierman. Online adaptive policy selection in time-varying systems: No-regret via contractive perturbations. In *Thirty-seventh Conference on Neural Information Processing Systems*, 2023.

[32] Matti Juhani Koivisto and Juan Pablo Murcia Leon. Solar PV generation time series (PECD 2021 update). *DTU-data*, 5 2022.

[33] Joseph Owen Roberts and Adam Warren. Us virgin islands wind resources update 2014. Technical report, National Renewable Energy Lab.(NREL), Golden, CO (United States), 2014.

[34] Mohammad Mahdian, Hamid Nazerzadeh, and Amin Saberi. Online optimization with uncertain information. *ACM Transactions on Algorithms (TALG)*, 8(1):1–29, 2012.

[35] Dhruv Rohatgi. Near-optimal bounds for online caching with machine learned advice. In *Proceedings of the Fourteenth Annual ACM-SIAM Symposium on Discrete Algorithms*, pages 1834–1845. SIAM, 2020.

[36] Thodoris Lykouris and Sergei Vassilvitskii. Competitive caching with machine learned advice. *Journal of the ACM (JACM)*, 68(4):1–25, 2021.

[37] Sungjin Im, Ravi Kumar, Aditya Petety, and Manish Purohit. Parsimonious learning-augmented caching. In *International Conference on Machine Learning*, pages 9588–9601. PMLR, 2022.

[38] Antonios Antoniadis, Christian Coester, Marek Elias, Adam Polak, and Bertrand Simon. Online metric algorithms with untrusted predictions. In *International Conference on Machine Learning*, pages 345–355. PMLR, 2020.

[39] Etienne Bamas, Andreas Maggiori, and Ola Svensson. The primal-dual method for learning augmented algorithms. *arXiv preprint arXiv:2010.11632*, 2020.

[40] Keerti Anand, Rong Ge, Amit Kumar, and Debmalya Panigrahi. Online algorithms with multiple predictions. In *International Conference on Machine Learning*, pages 582–598. PMLR, 2022.

[41] Pengfei Li, Jianyi Yang, Adam Wierman, and Shaolei Ren. Robust learning for smoothed online convex optimization with feedback delay. *Advances in Neural Information Processing Systems*, 36, 2024.

[42] Tongxin Li, Yiheng Lin, Shaolei Ren, and Adam Wierman. Beyond black-box advice: learning-augmented algorithms for mdps with q-value predictions. *Advances in Neural Information Processing Systems*, 36, 2024.

[43] Jianyi Yang, Pengfei Li, Tongxin Li, Adam Wierman, and Shaolei Ren. Anytime-competitive reinforcement learning with policy prior. *Advances in Neural Information Processing Systems*, 36, 2024.

[44] Yiheng Lin, Yang Hu, Guannan Qu, Tongxin Li, and Adam Wierman. Bounded-regret mpc via perturbation analysis: Prediction error, constraints, and nonlinearity. *Advances in Neural Information Processing Systems*, 35:36174–36187, 2022.

[45] Tongxin Li, Ruixiao Yang, Guannan Qu, Yiheng Lin, Adam Wierman, and Steven H Low. Certifying black-box policies with stability for nonlinear control. *IEEE Open Journal of Control Systems*, 2:49–62, 2023.

[46] Tongxin Li, Bo Sun, Yue Chen, Zixin Ye, Steven H Low, and Adam Wierman. Learning-based predictive control via real-time aggregate flexibility. *IEEE Transactions on Smart Grid*, 12(6):4897–4913, 2021.

[47] Nicolas Christianson, Christopher Yeh, Tongxin Li, Mahdi Torabi Rad, Azarang Golmohammadi, and Adam Wierman. Robustifying machine-learned algorithms for efficient grid operation. In *NeurIPS 2022 Workshop on Tackling Climate Change with Machine Learning*, 2022.

[48] Tongxin Li. *Learning-Augmented Control and Decision-Making: Theory and Applications in Smart Grids*. PhD thesis, California Institute of Technology, 2023.

[49] Tongxin Li and Chenxi Sun. Out-of-distribution-aware electric vehicle charging. *IEEE Transactions on Transportation Electrification*, 2024.

[50] Runyu Zhang, Yingying Li, and Na Li. On the regret analysis of online lqr control with predictions. *arXiv preprint arXiv:2102.01309*, 2021.

[51] Jean-Jacques E Slotine, Weiping Li, et al. *Applied nonlinear control*, volume 199. Prentice hall Englewood Cliffs, NJ, 1991.

[52] Yasin Abbasi-Yadkori and Csaba Szepesvári. Regret bounds for the adaptive control of linear quadratic systems. In *Proceedings of the 24th Annual Conference on Learning Theory*, pages 1–26. JMLR Workshop and Conference Proceedings, 2011.

[53] Max Simchowitz and Dylan Foster. Naive exploration is optimal for online lqr. In *International Conference on Machine Learning*, pages 8937–8948. PMLR, 2020.

[54] Sarah Dean, Horia Mania, Nikolai Matni, Benjamin Recht, and Stephen Tu. On the sample complexity of the linear quadratic regulator. *Foundations of Computational Mathematics*, pages 1–47, 2019.

[55] Naman Agarwal, Brian Bullins, Elad Hazan, Sham Kakade, and Karan Singh. Online control with adversarial disturbances. In *International Conference on Machine Learning*, pages 111–119. PMLR, 2019.

[56] John Doyle, Keith Glover, Pramod Khargonekar, and Bruce Francis. State-space solutions to standard $h_2$ and $h_\infty$ control problems. In *1988 American Control Conference*, pages 1691–1696. IEEE, 1988.

[57] Kemin Zhou and John Comstock Doyle. *Essentials of robust control*, volume 104. Prentice hall Upper Saddle River, NJ, 1998.

[58] Alberto Bemporad and Manfred Morari. Robust model predictive control: A survey. In *Robustness in identification and control*, pages 207–226. Springer, 1999.

[59] Pierre Comon. Independent component analysis, a new concept? *Signal processing*, 36(3):287–314, 1994.

[60] Aapo Hyvärinen and Petteri Pajunen. Nonlinear independent component analysis: Existence and uniqueness results. *Neural networks*, 12(3):429–439, 1999.

[61] Sébastien Lachapelle, Pau Rodriguez, Yash Sharma, Katie E Everett, Rémi Le Priol, Alexandre Lacoste, and Simon Lacoste-Julien. Disentanglement via mechanism sparsity regularization: A new principle for nonlinear ica. In *Conference on Causal Learning and Reasoning*, pages 428–484. PMLR, 2022.

[62] Aapo Hyvarinen, Hiroaki Sasaki, and Richard Turner. Nonlinear ica using auxiliary variables and generalized contrastive learning. In *The 22nd International Conference on Artificial Intelligence and Statistics*, pages 859–868. PMLR, 2019.

[63] Aapo Hyvärinen, Ilyes Khemakhem, and Hiroshi Morioka. Nonlinear independent component analysis for principled disentanglement in unsupervised deep learning. *Patterns*, 4(10), 2023.

[64] Yingying Li, Xin Chen, and Na Li. Online optimal control with linear dynamics and predictions: algorithms and regret analysis. In *NeurIPS*, pages 14858–14870, 2019.

[65] Wen-Hua Chen, Donald J Ballance, Peter J Gawthrop, and John O'Reilly. A nonlinear disturbance observer for robotic manipulators. *IEEE Transactions on industrial Electronics*, 47(4):932–938, 2000.

[66] Mesut E Baran and Felix F Wu. Network reconfiguration in distribution systems for loss reduction and load balancing. *IEEE Transactions on Power delivery*, 4(2):1401–1407, 1989.

[67] Na Li, Guannan Qu, and Munther Dahleh. Real-time decentralized voltage control in distribution networks. In *2014 52nd Annual Allerton Conference on Communication, Control, and Computing (Allerton)*, pages 582–588. IEEE, 2014.

[68] Masoud Farivar, Lijun Chen, and Steven Low. Equilibrium and dynamics of local voltage control in distribution systems. In *52nd IEEE Conference on Decision and Control*, pages 4329–4334. IEEE, 2013.

[69] Navin Goyal, Santosh Vempala, and Ying Xiao. Fourier pca and robust tensor decomposition. In *Proceedings of the forty-sixth annual ACM symposium on Theory of computing*, pages 584–593, 2014.

[70] Mikhail Belkin, Luis Rademacher, and James Voss. Blind signal separation in the presence of gaussian noise. In *Conference on Learning Theory*, pages 270–287. PMLR, 2013.

[71] Santosh S Vempala and Ying Xiao. Max vs min: Tensor decomposition and ica with nearly linear sample complexity. In *Conference on Learning Theory*, pages 1710–1723. PMLR, 2015.

[72] Diederik P Kingma and Jimmy Ba. Adam: A method for stochastic optimization. *arXiv preprint arXiv:1412.6980*, 2014.

[73] Joshua S Richman and J Randall Moorman. Physiological time-series analysis using approximate entropy and sample entropy. *American journal of physiology-heart and circulatory physiology*, 278(6):H2039–H2049, 2000.

[74] Peter Auer, Thomas Jaksch, and Ronald Ortner. Near-optimal regret bounds for reinforcement learning. *Advances in neural information processing systems*, 21, 2008.

[75] Mohammad Gheshlaghi Azar, Ian Osband, and Rémi Munos. Minimax regret bounds for reinforcement learning. In *International Conference on Machine Learning*, pages 263–272. PMLR, 2017.

[76] Elad Hazan. *Introduction to online convex optimization*. MIT Press, 2022.

**Broader Impacts.** As demonstrated in our experiments (see Appendix C), the ability to incorporate and correctly utilize latent variable predictions in control systems can drastically improve decision-making processes in critical areas such as autonomous driving, aerospace navigation, and healthcare systems, leading to more efficient and robust system performance under varying conditions. Despite that by leveraging disentangled predictions, our approach can potentially reduce the need for extensive data in making accurate predictions, concerns exist when implementing our method in real systems. For example, in the voltage control task, disentangling power generation resources including the residential consumption, solar injection, wind generation, etc may lead to discrimination among the system users.

# A  Related Work

Our work contributes to the growing community of algorithms with predictions, while also incorporating ideas from adaptive control, online learning, and disentangled representation learning.

**Online Decision-Making with Predictions.** The challenge posed by untrusted ML advice in online decision-making processes has attracted increasing attention, as evidenced by recent studies (e.g., [34, 18]) for various models such as ski rental [18, 19, 20], caching [35, 36, 37], bipartite matching [38], online covering [39, 40], online optimization [21, 41], reinforcement learning [42, 43], control [3, 4, 44, 45], and real-world applications [46, 47, 48, 49] with focuses on exploring the implications and potential solutions for handling unreliable ML predictions. The most relevant result to this work is [4], which establishes a consistency-robustness tradeoff for the linear quadratic control problem with untrusted predictions. However, the self-tuning algorithm proposed in [4] has a competitive ratio that suffers from high variability of system perturbations and predictions, yielding a sub-optimal consistency and robustness tradeoff provided there.

**Linear Quadratic Control** Our work intersects with and extends upon established research in the area of linear quadratic control (LQC) systems and the analysis of the linear quadratic regulator (LQR), particularly in the context of integrating predictions with control strategies. The existing literature primarily explores the balance between achieving optimal control and managing the uncertainties inherent in predictive models. Significant work has been conducted in the realm of LQR systems, especially regarding model predictive control (MPC). For instance, Yu et al. in [2] have analyzed LQR regret in MPC under the assumption of accurate perturbation predictions. The implications of imprecise predictions are explored in [22]. In addition, there are notable contributions to the discourse on regret and competitive ratio in MPC, as evidenced by works like [1, 50, 4]. Our work is closely related to [4], which reveals a consistency and robustness tradeoff in LQC, with a self-tuning policy that updates a 1-dimensional confidence parameter based on past observations.

**Adaptive and Robust Control** Moreover, our approach also aligns with the adaptive control paradigm, especially in its recent intersection with learning theory to address non-asymptotic metrics. While the adaptive control community has traditionally concentrated on Lyapunov stability and asymptotic convergence [51], the newer trend represented in [52, 53, 54, 55] employs learning-theoretic measures such as regret and dynamic regret for finite-time horizon analysis. Our results contribute to this evolving field by presenting a adaptive control policy that achieves the first of-its-kind near-optimal consistency and robustness tradeoff, as a novel endeavor in the context of LQR systems that also incorporates the unique challenge raised by untrusted predictions. Besides, robust control is a large area that concerns the design of controllers with performance guarantees that are robust against model uncertainty or adversarial disturbances [12]. Tools of robust control include $H_\infty$ synthesis [56, 57] and robust MPC [58]. Our work diverges from these existing approaches by considering an LQC model that employs disentangled predictions. While it adheres to the robust control principle of resilience to uncertainties and perturbations, our work also aims to achieve near-optimal performance when predictions become accurate. This best-of-both-worlds strategy in utilizing untrusted predictions introduces a novel dimension to the robustness-consistency tradeoff, a concept not extensively explored in the existing robust control literature.

**Learning Disentangled Latent Variables** Disentanglement, a key goal of which is to separate the typically independent effects of latent variables on observations, is a well-established concept within the field of representation learning. For instance, when the underlying mixing function that

mixes latent independent signals to observations is linear with respect to $s_t$, i.e., $f(s_t; \theta) = \theta s_t$, for a matrix $\theta \in \mathbb{R}^{n \times k}$, linear independent component analysis (ICA) [59, 30] can be effectively employed to identify the $k$ latent components. This linear approach, while powerful in its simplicity, is often insufficient for more complex, real-world data structures. In contrast, advanced nonlinear ICA methods [60] have been developed to address these complexities, providing more nuanced tools for disentangling latent perturbations. These methods extend the traditional ICA framework to accommodate nonlinear mixing functions, thereby enhancing its applicability to a wider range of scenarios. State-of-the-art techniques in nonlinear ICA have introduced novel algorithms that can effectively manage the intricacies of nonlinear relationships between observed and latent variables. Furthermore, nonlinear ICA has been explored under various settings, with multiple assumptions ensuring the identifiability of the model, such as sparsity [61, 17], auxiliary information [62, 15], and others. Recent work by Hyvärinen et al. [63] has utilized contrastive learning to identify the nonlinear ICA model in a time-series context, demonstrating the evolving capabilities of these methods. Table 1 in Section B provides a summary of key assumptions of recent nonlinear ICA methods. Note that these methods often incorporate deep neural networks as critical black-box elements within their algorithmic structures. We focus on optimally harnessing the predictions of latent variables derived from those aforementioned disentanglement methods, regardless of their trustworthiness. Therefore, our work uniquely represents a strategic shift towards effectively utilizing uncertain or unverified predictions in downstream decision-making tasks, a challenge not directly addressed by existing representation learning techniques.

## B Examples and Applications

### B.1 Latent Perturbation Modelling

Understanding how the observed variables relate to a set of latent variables $s_t = (s_t(1), \ldots, s_t(k))$ has been extensively studied in machine learning. When the mixing function $f$ is linear, standard statistical assumptions (see Corollary B.1) on $s_t$ can guarantee the desired identifiability of $f$, up to multiplicative scaling and permutation. For the case when $f$ is nonlinear, recent advances such as [13, 14, 15, 16, 17] in nonlinear independent component analysis (ICA) leverage additional assumptions on the latent variables in $s_t$ and the mixing function $f$, to make the model identifiable. Below in Table 1 we summarize the key assumptions (besides the common assumptions of mutual independence between latent variables and at most one latent variable is Gaussian) made in those models.

Table 1: Standard assumptions on $f$ and $s$ for a subset of nonlinear ICA models.

| Nonlinear ICA Models | Key Assumptions on $f$ |
| --- | --- |
| Identifiable VAE [15] | Mixing function $f$ is bijective and smooth |
| Contrastive learning [13, 14] | Mixing function $f$ is bijective and smooth |
| Structural sparsity model [17] | Support of the Jacobian $\mathsf{J}_f(s)$ of $f$ is sparse |
| Volume-preserving model [16] | Mixing function $f$ is bijective and $|\det \mathsf{J}_f(s)| = 1$ |

| Nonlinear ICA Models | Key Assumptions on $s$ |
| --- | --- |
| Identifiable VAE [15] | $(s_t(1), \ldots, s_t(k))$ are conditionally independent given a variable $u$ |
| Contrastive learning [13, 14] | $(s_t : t \in [T])$ is nonstationary or has temporal dependencies |

### B.2 Real-World Applications

Our model represents a classical control framework with extensive applicability across a variety of real-world decision-making problems. In the following section, we illustrate the versatility of our model through a selection of examples that demonstrate our dynamical model in (2) (see Section 2.1), subject to latent perturbations. These applications not only exemplify our model, but also set the stage for the comprehensive experimental results we discuss later in Section C, where we validate the model's practical efficacy in real-world scenarios.

**Example 1** (Piloting a Drone with mixed disturbances). *Imagine the challenge of piloting a drone to follow an unpredictable path $(y_t \in \mathbb{R}^2 : t \in [T])$ on a windy and rainy day. This dynamic scenario*

*can be mathematically represented in a discrete-time counterpart of the kinematic model [64, 2]:*

$$\begin{bmatrix} \Delta x_{t+1} \\ v_{t+1} \end{bmatrix} = A \begin{bmatrix} \Delta x_t \\ v_t \end{bmatrix} + Bu_t + Cw_t,$$

*for some matrices $A \in \mathbb{R}^{4 \times 4}$, $B \in \mathbb{R}^{4 \times 2}$, $C \in \mathbb{R}^{4 \times 4}$. The column vector $w_t = (w_t(1), w_t(2), w_t(3), w_t(4))$ contains 4 latent variables where $w_t(1) := y_t(1) - y_{t+1}(1)$; $w_t(2) := y_t(2) - y_{t+1}(2)$; $w_t(3)$ is the environmental wind speed and $w_t(4)$ is the rainfall intensity at time $t \in [T]$. More details are discussed in Section 5.1.*

Besides, it is well-known that any controllable system can be transformed into a canonical linear input-disturbed systems with dynamics described by $x_{t+1} = Ax_t + B(u_t + w_t)$ [65, 1] that can be used in many real-world applications with disturbances added to the control inputs. Now, we introduce another example in an electricity grid, with latent variables representing power injections from distinct buses of a distribution network.

***Example* 2** (Heterogeneous Voltage Control). *Consider a Volt/Var control task with the Simplified Distflow model in a distribution power network consisting of $n$ branch buses and a substation bus [66, 67]:*

$$v_{t+1} = Xq_t^c + Gp_t + Xq_t^e + v_0 \mathbf{1}_n, \tag{15}$$

*wherein imaginary constants of the complex power injection at time $t \in [T]$ are separated into two parts $q_t^c \in \mathbb{R}^n$ and $q_t^e \in \mathbb{R}^n$, representing the controllable reactive power injection and the uncontrollable external injection (perturbation) respectively. The fixed and known matrices $X, G \in \mathbb{R}^{n \times n}$ are positive definite and positive (Lemma 1 in [68]) with their entries depend on the line impedance and the grid topology. Our goal is to design a Volt/Var controller to regulate squared voltage magnitudes $v_{t+1} = (v_{t+1}(1), \dots, v_{t+1}(n))$ to their nominal values by provisioning the reactive power injection $q_t^c$ for each time step $t \in [T]$.*

For the above example, we focus on a scenario with a set of $n$ heterogeneous users corresponding to $n$ branch nodes that generate a sequence of time-varying active and reactive injections $(p_t(i) : i = 1, \dots, n)$ and $(q_t^c(i) : i = 1, \dots, n)$. The injection at the branch nodes $i = 1, \dots, n$ may be generated by various types of renewable resources and power consumption patterns, e.g., solar/wind generations, residential, and commercial activities, energy storage management, or charging EVs, etc. Together they form a set of latent variables $s_t$ in (2) with $k = n$ and in this special case, the mixing function $f(s_t; \theta) = Gp_t + Xq_t^e + v_0 \mathbf{1}_n$ is affine.

In Section 5.1 and 5.2, we explore in greater detail the two examples previously discussed, and demonstrate the efficacy of our proposed method through case studies using real-world data.

## B.3 Sample efficient ICA

Furthermore, there are sample efficient ICA algorithms for independent latent variables [69, 70, 71]. The following corollary can be derived from Theorem 4.2 and the sample complexity result in [71], which shows that $\overline{\eta}$ is sub-linear in $T$, thereby concluding the best-of-both-worlds prediction utilization claimed in Section 1.

***Corollary* B.1.** *Suppose the mixing matrix $\theta$ is full column rank, and the latent variables $(s_t(i) : i = 1, \dots, k)$ are mutually independent for all $t \in [T]$. Let $w = \Theta(\log T)$. Under the classic assumptions such that $\mathbb{E}[s_t(i)] = 0$, $\mathbb{E}[s_t^2(i)] = 1$, $\mathbb{E}[|s_t^5(i)|] \leq M$, and the fourth cumulant $|\mathrm{cum}_4(s_t(i))| \geq \Delta$ for all $t \in [T]$ and $i = 1, \dots, k$, there exists a polynomial-time ICA algorithm that provides a sequence of mixing matrix estimates $(\widetilde{\theta}_t : t \in [T])$ of $\theta$ such that with high probability, the competitive ratio of* DISC *satisfies*

$$\mathsf{CR}(\mathrm{DISC}) \leq 1 + O\left( \sum_{i=1}^{k} \frac{\overline{\varepsilon}(i)}{T/\log T + \overline{\varepsilon}(i)} \right) + O\left( \log T \sqrt{\frac{k}{T}} \right) + O\left( \rho_F^{2w} + (\log n)^{\frac{7}{2}} k^{\frac{1}{2}} \frac{\log T}{T} \right).$$

Therefore, if for all $i \in \{1, \dots, k\}$, $\overline{\varepsilon}(i)$ is negligible, $\mathsf{CR}(\mathrm{DISC})$ will converge to the optimal competitive ratio 1 with an increasing time horizon length $T$ and a sufficiently large prediction window $w$ (e.g., $w = \Theta(\log T)$); otherwise, for any $\overline{\varepsilon}(i) > 0$, $\mathsf{CR}(\mathrm{DISC})$ will be asymptotically bounded from above by $O(k) + 1$ where $k$ is the number of latent variables. It is worth noting that the statistical assumptions on the latent variables $(s_t(i) : i = 1, \dots, k)$ are classic in the ICA analysis [69, 70, 71].

# C  Case Studies

In this section, we delve deeper into the practical applications of our main results by using two real-world examples (see Example 1 and 2) to demonstrate the efficacy of DISC. We first describe our experimental setup in Section 5. We then demonstrate the effectiveness of the proposed method in Section 5.1, and compare DISC with several baselines. In Section 5.2, we show the desired resilience with respect to a non-stationary environment in a voltage control task.

**Online Learning of Confidence Parameters.**  We illustrate the convergence of the confidence parameter $(\lambda_t : t \in [T])$ with respect the latent dimensions. In Figure 4, we exemplify the convergence behavior of the confidence parameters associated with each of the four latent components shown in (16) on the left column, together the time series for the four latent components on the right.

Notably, the first three components display learnable patterns while the last component, representing rainfall intensity is some unpredictable Gaussian noise. This distinction is reflected in the convergence behaviors of the corresponding confidence parameters: $\lambda(1)$, $\lambda(2)$, and $\lambda(3)$, associated with the more predictable components, quickly converge to the optimal value of 1. Conversely, the last parameter $\lambda(4)$ associated with the erratic rainfall component exhibits a decreasing trend due to the unpredictability of the fourth component, highlighting the resilience of the proposed DISC policy against the prediction error.

## C.1  Implementation Details of Experiment A

The drone's location at each moment, $p_{t+1} \in \mathbb{R}^2$, is determined by its current position and velocity $v_t \in \mathbb{R}^2$, evolving as $p_{t+1} = p_t + c_p v_t$. The drone controller can adjust its velocity at every time step through a control input $u_t$, leading to the updated velocity $v_{t+1} = v_t + c_v u_t$. The coefficients $c_p, c_v > 0$ are determined by control time intervals and physical parameters like the drone's weight and the drag coefficient. Defining $\Delta x_t := p_t - y_t$ the location mismatch at time $t$, this piloting task can be mathematically represented in a linear control framework as shown in (2):

$$\mathbb{R}^4 \ni \begin{bmatrix} \Delta x_{t+1} \\ v_{t+1} \end{bmatrix} = A \begin{bmatrix} \Delta x_t \\ v_t \end{bmatrix} + B u_t + \underbrace{\begin{bmatrix} y_t \\ \mathbf{0}_{2 \times 1} \end{bmatrix} - \begin{bmatrix} y_{t+1} \\ \mathbf{0}_{2 \times 1} \end{bmatrix} + w_t c_1 + r_t c_2}_{\text{4 latent components}}, \qquad (16)$$

with system matrices $A := \begin{bmatrix} 1 & 0 & c_p & 0 \\ 0 & 1 & 0 & c_p \\ 0 & 0 & 1 & 0 \\ 0 & 0 & 0 & 1 \end{bmatrix}, B := \begin{bmatrix} 0 & 0 \\ 0 & 0 \\ c_v & 0 \\ 0 & c_v \end{bmatrix}.$

The term $w_t c_1 + r_t c_2$ represents external perturbations from the environment. The column vectors $c_1, c_2 \in \mathbb{R}^4$ quantify the effects of the stochastic time-varying wind speed and rainfall intensity, represented by two time series $(w_t : t \in [T])$ and $(r_t : t \in [T])$. We assume the wind speed time series forms a sinusoidal function and the rain intensity time series is Gaussian, which are visualized together with the ML predictions in Figure 4. Table 2 in Appendix C.3 presents detailed choices of the matrices $A, B, Q, R$ and vectors $c_1, c_2$.

## C.2  Additional Experimental Results

We provide further experimental results that complement and extend the results presented in Section C.

In Figure 5, we display the trajectories in the drone navigation task (Section 5.1) corresponding to DISC and the three baseline policies: the LQR, the self-tuning policy in [4], and the MPC with untrusted predictions (setting $\lambda = \mathbf{1}_k$ in (7)). The trajectory for DISC closely aligns the best with the offline optimal trajectory upon convergence.

In Figure 6, for the voltage control task (see Section 5.2), we compare the average total costs $J(\pi)$ (defined in (3)) corresponding to DISC and the three baseline policies. Note that DISC consistently achieves the lowest average cost with the smallest variance computed from 5 random tests. The scaling parameter $\nu$ of the Gaussian component at node 8 is set as 1.25.

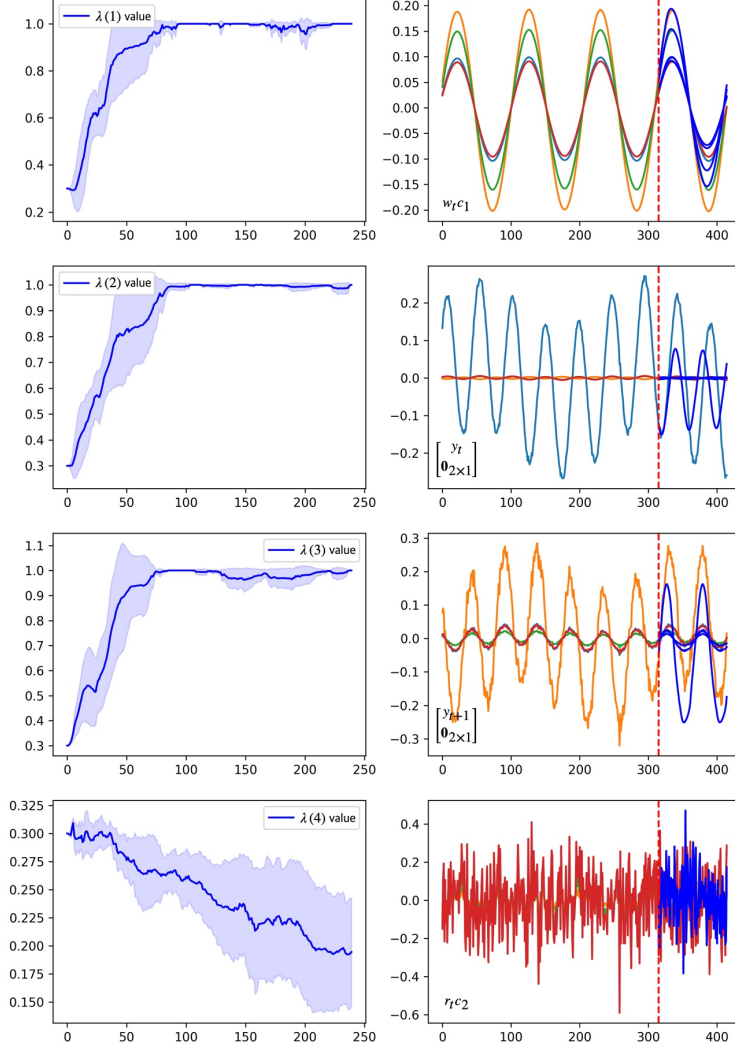

Figure 4: **Example A.** *Drone Navigation (Section 5.1).* **Left column**: *Convergence of confidence parameters* $\lambda(1), \lambda(2), \lambda(3), \lambda(4)$ *corresponding to* 4 *latent components.* **Right column**: *Temporal dynamics of latent components* $\theta s_t$ *in* $\mathbb{R}^4$. We illustrate the time series for the latent components in (16), with each sub-figure featuring four distinct curves representing the dimensions of the respective 4-dimensional component. Post the dotted red line, the additional blue curves visualize imperfect ML predictions from the multi-layer neural network, elaborated in Appendix C.3. The $x$-axis in each graph marks the time steps, while the $y$-axis denotes the values of the time series.

## C.3 Detailed Setup and Hyper-Parameters

**ML Model Setup**    We train an ML model online to generate predictions of $(s_\tau(i)\theta(i) : t \le \tau \le \bar{t})$ at each time $t \in [T]$. Here, $\theta(i)$ denotes the $i$-th column of the true mixing parameter matrix $\theta$. At each time step, the prediction model is trained and updated using previously collected disturbances. The warm-start buffer size is set as 100 and 50 respectively for the drone navigation and voltage control applications. In particular, we update the ML prediction model at each time step $t \in [T]$ and use a length-$b$ subsequence of the collected time series $(s_\tau(i)\widetilde{\theta}_t(i) : \max\{0, \tau - b\} \le \tau < t)$ as the input to predict the future $w = 5$ ($w$ is the prediction window size defined in Section 2) steps of perturbations, as the output. Note that $\widetilde{\theta}_t(i)$ is the $i$-th column of the estimated mixing parameter matrix $\widetilde{\theta}_{t-1}$ provided by the FastICA method in [30] at time $t \in [T]$. All prediction models are formed via a fully-connected neural network with 4 hidden layers with a width 80 and LeakyReLU as the activation function except the final layer, and are trained using Adam [72] as the optimizer with a learning rate 1e$-3$ for 500 epochs.

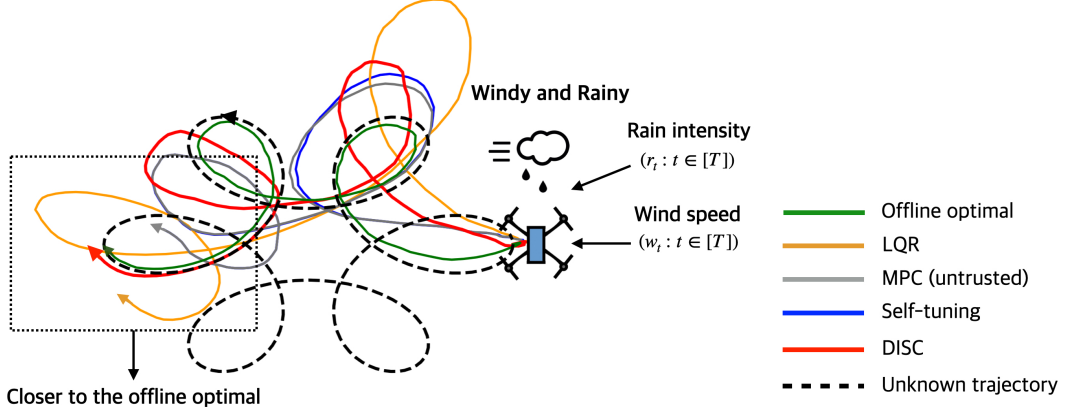

Figure 5: Supplementary results showcasing drone navigation in windy and rainy conditions, further illustrating the scenarios discussed in Section 5.1 depicted in Figure 2. The external perturbations impacting the drone's flight are modeled by two independent, time-varying latent variables: $w_t$ for wind speed and $r_t$ for rain intensity at time $t \in [T]$. The path navigated by the drone using the DISC policy (detailed in Algorithm 1 in Section 3) is traced in red. Notably, this trajectory closely aligns the best with the offline optimal trajectory upon convergence, demonstrating the effectiveness of the DISC policy in adapting to complex environmental conditions.

**Disentanglement Implementation** A well-recognized issue in independent component analysis (ICA) and its nonlinear variants is the permutation indeterminacy of sources. This implies that the ordering of disentangled components can vary, such that the $i$-th component identified at one time step may not correspond to the $i$-th component in subsequent algorithm outputs. To mitigate this "permutation unidentifiability", our methodology involves pre-processing the time series data by ranking the components according to their sample entropy, as suggested in [73]. This step precedes the application of the ML prediction model, thereby enhancing the consistency of component identification across time steps.

Table 2 below summarizes the detailed parameters used in Section C. In our experiments, the parameters are not subject to extensive optimization.

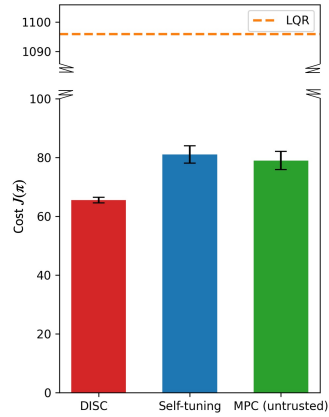

Figure 6: Average total cost $J(\pi)$ with error bars for four policies in the voltage control task (Section 5.2).

### C.4 Detailed Setup in Section 5.2

The matrices $X$ and $G$ are generated from the following an impedance matrix $Z \in \mathbb{C}^{11 \times 11}$ with the following non-zero entries:

$$Z_{0,1} = Z_{1,0} = Z_{1,2} = Z_{2,1} = 1 + i,$$
$$Z_{2,3} = Z_{3,2} = Z_{2,4} = Z_{4,2} = 1 + 2i,$$
$$Z_{5,9} = Z_{9,5} = Z_{5,10} = Z_{10,5} = 2 + 4i,$$
$$Z_{3,5} = Z_{5,3} = Z_{3,6} = Z_{6,3} = Z_{4,7} = Z_{7,4} = Z_{4,8} = Z_{8,4} = 2 + 2i,$$

where $i$ denotes the imaginary unit.

| Parameter | Value |
|---|---|
| Time horizon length $T$ | 240 (Section 5.1), 200 (Section 5.2) |
| Warm start buffer length | 100 (Section 5.1), 50 (Section 5.2) |
| State dimension $n$ | 4 (Section 5.1), 10 (Section 5.2) |
| Action dimension $m$ | 2 (Section 5.1), 10 (Section 5.2) |
| Prediction window size $w$ | 5 |
| Coefficients $c_v, c_p$ | 0.2 |
| Mixing Column $c_1$ | $\frac{1}{8}(0.8, 1.6, 1.25, 0.75)^\top$ |
| Mixing Column $c_2$ | $\frac{1}{8}(1.2, 0.4, 0.75, 1.25)^\top$ |
| $Q, R$ (Section 5.1) | $\begin{bmatrix} I_{2\times2} & \mathbf{0}_{2\times2} \\ \mathbf{0}_{2\times2} & \mathbf{0}_{2\times2} \end{bmatrix}, I_{2\times2}$ |
| $Q, R$ (Section 5.2) | $I_{10\times10}, 0.1 \times I_{10\times10}$ |
| Initial confidence parameter $\lambda_0$ | 0.3 |

(a) Basic Control Problem Setup in Section 5.1 and 5.2.

| Hyper-Parameter | Value |
|---|---|
| Number of hidden layers | 4 |
| Hidden layer width | 80 |
| Covariate size $b$ | 5 |
| Input dimension | $n \times b$ |
| Output dimension | $n \times w$ |
| Activation function | LeakyReLU |
| Optimizer | Adam [72] |
| Learning rate | 1e−3 |

(b) Neural Network Hyper-parameters.

Table 2: Parameters and hyper-parameters used in Section C.

# D  Useful Lemmas

We first present some preliminary results that will be used when proving our main theorems.

## D.1  Convexity of $\zeta_\ell$

Below we verify that $\zeta_\ell(\lambda)$ is a convex function of $\lambda \in [0,1]^k$. By definition,

$$\left[ \sum_{\tau=\underline{\ell}}^{\ell} (F^\top)^{\ell-\tau} P\left(\theta s_\ell - \widetilde{\theta}_\tau(\lambda \circ \widetilde{s}_{\ell|\tau})\right) \right]^\top H \left[ \sum_{\tau=\underline{\ell}}^{\ell} (F^\top)^{\ell-\tau} P\left(\theta s_\ell - \widetilde{\theta}_\tau(\lambda \circ \widetilde{s}_{\ell|\tau})\right) \right] \tag{17}$$

equals to (since $H$ is symmetric)

$$\underbrace{\left[ \sum_{\tau=\underline{\ell}}^{\ell} (F^\top)^{\ell-\tau} P\theta s_\ell \right]^\top H \left[ \sum_{\tau=\underline{\ell}}^{\ell} (F^\top)^{\ell-\tau} P\theta s_\ell \right]}_{\text{Constant independent of } \lambda}$$

$$+ \left[ \sum_{\tau=\underline{\ell}}^{\ell} (F^\top)^{\ell-\tau} P\widetilde{\theta}_\tau(\lambda \circ \widetilde{s}_{\ell|\tau}) \right]^\top H \left[ \sum_{\tau=\underline{\ell}}^{\ell} (F^\top)^{\ell-\tau} P\widetilde{\theta}_\tau(\lambda \circ \widetilde{s}_{\ell|\tau}) \right]$$

$$-2 \left[ \sum_{\tau=\underline{\ell}}^{\ell} (F^\top)^{\ell-\tau} P\theta s_\ell \right]^\top H \left[ \sum_{\tau=\underline{\ell}}^{\ell} (F^\top)^{\ell-\tau} P\widetilde{\theta}_\tau(\lambda \circ \widetilde{s}_{\ell|\tau}) \right].$$

Now, we simplify the remaining two terms. We first notice that $\sum_{\tau=\underline{\ell}}^{\ell}(F^\top)^{\ell-\tau}P\widetilde{\theta}_\tau(\lambda \circ \widetilde{s}_{\ell|\tau}) = \Lambda\lambda$ for some matrix $\Lambda \in \mathbb{R}^{n\times k}$. To see this, denote $\phi_{\ell,\tau} := (F^\top)^{\ell-\tau}P\widetilde{\theta}_\tau \in \mathbb{R}^{n\times k}$. We get

$$\sum_{\tau=\underline{\ell}}^{\ell}(F^\top)^{\ell-\tau}P\widetilde{\theta}_\tau(\lambda \circ \widetilde{s}_{\ell|\tau}) = \sum_{\tau=\underline{\ell}}^{\ell}\phi_{\ell,\tau}(\lambda \circ \widetilde{s}_{\ell|\tau}) = \begin{bmatrix} \vdots & & \vdots \\ \phi_{\ell,\tau}(1) & \cdots & \phi_{\ell,\tau}(k) \\ \vdots & & \vdots \end{bmatrix}\begin{bmatrix} s_1\lambda_1 \\ \vdots \\ s_k\lambda_k \end{bmatrix}$$

$$= \underbrace{\begin{bmatrix} \vdots & & \vdots \\ s_1\phi_{\ell,\tau}(1) & \cdots & s_k\phi_{\ell,\tau}(k) \\ \vdots & & \vdots \end{bmatrix}}_{=:\Lambda}\begin{bmatrix} \lambda_1 \\ \vdots \\ \lambda_k \end{bmatrix}.$$

Therefore, it suffices to validate that the matrix $H$ is positive semi-definite to show $(\Lambda\lambda)^\top H(\Lambda\lambda) - 2\vartheta^\top H(\Lambda\lambda)$ is convex, where we denote $\vartheta := \sum_{\tau=\underline{\ell}}^{\ell}(F^\top)^{\ell-\tau}P\theta s_\ell \in \mathbb{R}^n$. Note that $H = B(R + B^\top PB)^{-1}B^\top$. Since $R$ is positive definite by assumption and the DARE solution $P$ is also positive definite, $(\Lambda\lambda)^\top H(\Lambda\lambda) - 2\vartheta^\top H(\Lambda\lambda)$ is a summation of a quadratic form and a linear function of $\lambda$, validating the convexity of the term in (17). For the same reason, $\zeta_\ell(\lambda) = \sum_{\tau=\underline{\ell}}^{\ell} g(\tau,\ell)(\lambda)^\top Hg(\tau,\ell)(\lambda)$ is also convex.

### D.2  A Generalized Lower Bound on $J^\star$

The following lemma is a generalized result from [4], which provides a lower bound on the offline optimal cost $J^\star$, as a function of system perturbations.

**Lemma 2** (Generalized Lower Bound on $J^\star$ [4])**.** *Let $\lambda_{\min}(Q)$ and $\lambda_{\min}(P)$ denote the smallest eigenvalues of positive definite matrices $Q$ and $P$, respectively. It follows that*

$$J^\star \geq C_0 \sum_{t=0}^{T-1}\left(\sum_{\tau=t}^{T-1}\rho_F^{\tau-t}\|f(s_\tau;\theta)\|\right)^2 \tag{18}$$

*for some constant $0 < C_0 \leq \frac{(1-\rho_F)^2}{2}\min\{\lambda_{\min}(P), \lambda_{\min}(R)/\|B\|, \lambda_{\min}(Q)/\max\{2,\|A\|\}\}$.*

Next, we present the proof of Lemma 1.

### D.3  Proof of Lemma 1

We restate the lemma below, which reduces the online learning objective $\sum_{\ell=0}^{t-1}\psi_{\ell,t}^\top(\lambda)H\psi_{\ell,t}(\lambda)$ to a canonical online optimization formulation.

**Lemma 3.** *Define $\zeta_\ell : \mathcal{I} \to \mathbb{R}^n$ as in (11). Then for any $t \in [T]$, it follows that*

$$\sum_{\ell=0}^{t-1}\psi_{\ell,t}^\top(\lambda)H\psi_{\ell,t}(\lambda) \leq w\sum_{\ell=0}^{t-1}\zeta_\ell(\lambda).$$

Denote by $g(\tau,\ell) := (F^\top)^{\tau-\ell}P\left(f(s_\tau;\theta_\tau) - f\left(\lambda \circ \widetilde{s}_{\tau|\ell};\widetilde{\theta}_\ell\right)\right)$. We first note that

$$\sum_{\ell=0}^{t-1}\sum_{\tau=\ell}^{\overline{\ell}}g(\tau,\ell)^\top Hg(\tau,\ell) = \sum_{\tau=0}^{t-1}\sum_{\ell=\tau}^{\overline{\tau}}g(\ell,\tau)^\top Hg(\ell,\tau) = \sum_{\tau=0}^{t-1}\zeta_\tau(\lambda). \tag{19}$$

Since $H = B(R + B^\top PB)^{-1}B^\top$ is positive semi-definite as we have shown in Appendix D.1, the function $h(x) := x^\top Hx$ is convex. Applying the Jensen's inequality,

$$h\left(\frac{1}{w}\sum_{\tau=\ell}^{\overline{\ell}}g(\tau,\ell)\right) \leq \frac{1}{w}\sum_{\tau=\ell}^{\overline{\ell}}h\left(g(\tau,\ell)\right).$$

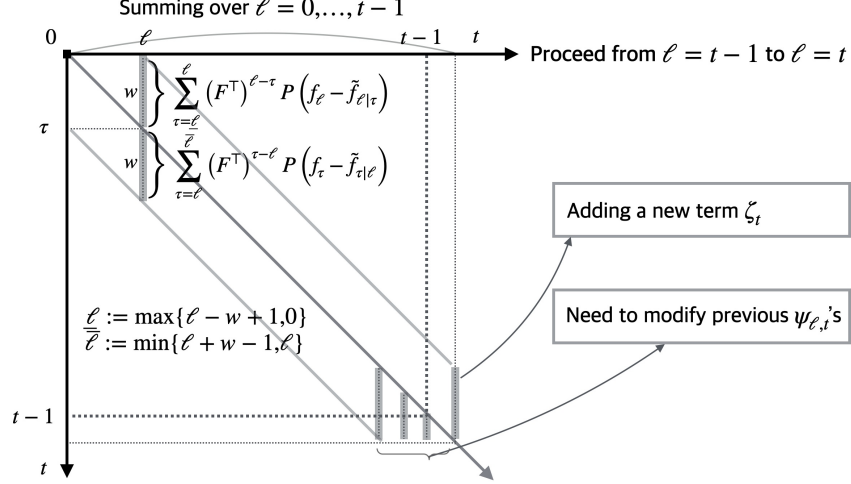

Figure 7: Illustration of the equivalence in (19). When $\ell$ increases from $t-1$ to $t$, the reduced form adds a new term $\zeta_t$ while the original form needs to update $w$ previous functions $\psi_{\ell,t}$ for $\ell = t - w, \ldots, t - 1$.

Therefore,

$$\sum_{\ell=0}^{t-1} \psi_{\ell,t}^\top(\lambda) H \psi_{\ell,t}(\lambda) \le w \sum_{\ell=0}^{t-1} \sum_{\tau=\ell}^{\overline{\ell}} g^\top(\tau,\ell) H g(\tau,\ell) = w \sum_{\tau=0}^{t-1} \zeta_\tau(\lambda), \tag{20}$$

where (20) follows from (19). The proof is illustrated in Figure 7. We now proceed to show our main results. The key structure of the lemmas and theorems is outlined in Figure 8.

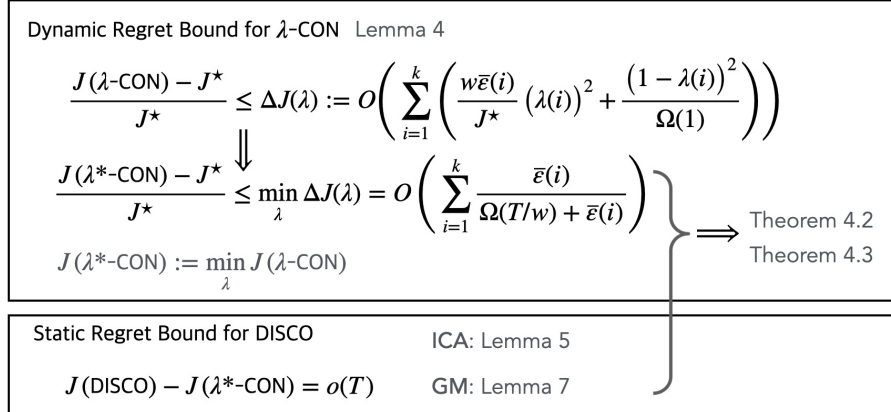

Figure 8: Outline of key steps in the proofs of Theorem 4.2 and 4.3. Arrows denote implications.

# E   Proof of Theorem 4.2

We first state a detailed version of Theorem 4.2 below with explicit multiplicative constants.

**Theorem E.1** (DISC for Linear Mixing). *Let $H := B(R + B^\top PB)^{-1}B^\top$. With our model assumptions, the competitive ratio of* DISC *with a linear mixing function satisfies*

$$\mathrm{CR}(\mathrm{DISC}) \le 1 + \frac{C_1}{J^\star}\left(2\overline{\eta} + \left(\overline{\theta}\rho_F^w\right)^2 T\right) + 8C_2 w \left(\sum_{i=1}^k \left(\frac{\overline{\varepsilon}(i)}{C_0 \sigma_{\min}^2(\theta)\overline{\varepsilon}(i) + J^\star}\right) + \left(\frac{\overline{s}^2}{J^\star}\sqrt{kT}\right)\right),$$

*where $\sigma_{\min}(\theta) > 0$ is the smallest singular value of $\theta$; $\overline{s}$ and $\overline{\theta}$ are defined in Section 4.2; $C_0$ is defined in Lemma 2; $C_1 := 2\|H\|\left(\frac{C_F}{1-\rho_F}\|P\|\overline{s}\right)^2$, and $C_2 := \|H\|\left(C_F\|P\|\overline{\theta}\right)^2$.*

We now proceed to prove Theorem E.1.

**Step 1: Consistency-Robustness Analysis of $\lambda$-Con**    We first show the following lemma, which highlights a tradeoff between consistency and robustness. The following definition of dynamic regret will be used as an alternative worst-case performance metric in our main results. Dynamic regret is a more general (and often more challenging to analyze) measure than classical static regret, which has been mostly used for stationary environments [74, 75].

**Definition E.1** (Dynamic regret). The *dynamic regret* of a policy $\pi = (\pi_t : t \in [T])$ is defined as the difference between the quadratic cost induced by the policy $\pi$, $J(\pi)$ in (3), and the offline optimal cost $J^\star := \inf_\pi J(\pi)$ subject to (2), i.e., $\mathrm{DR}(\pi) := J(\pi) - J^\star$.

**Lemma 4** (Consistency-Robustness Lemma). *With a fixed trust parameter $\lambda \in \mathcal{I}$, the disentangled $\lambda$-confident policy in Section 3.1 has a dynamic regret of at most*

$$2\|H\| \left(2\|P\|C_F \overline{s}\right)^2 \left(\overline{\eta} + \left(\overline{\theta}\rho_F^w\right)^2 T\right) + 8\|H\| \sum_{t=0}^{T-1} \left(g_t^{con} + g_t^{rob}\right), \tag{21}$$

*where $P, C_F, \overline{\eta}, \rho_F$ are defined in Section 2; $\overline{s}$ and $\overline{\theta}$ are defined in Section 4.2. The terms $g_t^{con}$ and $g_t^{rob}$ are functions of $\lambda$, defined as*

$$g_t^{con} := \left\| \sum_{\tau=t}^{\overline{T}} \left(F^\top\right)^{\tau-t} P\left(\widetilde{\theta}_t \left(\lambda \circ \varepsilon_{\tau|t}\right)\right) \right\|^2, \quad g_t^{rob} := \left\| \sum_{\tau=t}^{\overline{T}} \left(F^\top\right)^{\tau-t} P\left(\widetilde{\theta}_t \left((\mathbf{1}_k - \lambda) \circ s_{\tau|t}\right)\right) \right\|^2.$$

*Proof of Lemma 4.* Denote by $\lambda$-Con the $\lambda$-confident policy, and $J(\lambda\text{-Con})$ the corresponding total cost induced by taking actions $(u_0, \ldots, u_{T-1})$ generated by $\lambda$-Con. Similarly, denote by $J^\star$ the optimal total cost. Denote $\overline{T} := \min\{t + w - 1, T - 1\}$. Lemma 3 in [4] implies the following explicit form of $J(\lambda\text{-Con}) - J^\star$:

$$\sum_{t=0}^{T-1} \left(\sum_{\tau=t}^{T-1} \left(F^\top\right)^{\tau-t} P\left(\theta_\tau s_\tau - \kappa(\tau)\widetilde{\theta}_t \left(\lambda \circ \widetilde{s}_{\tau|t}\right)\right)\right)^\top H$$
$$\cdot \left(\sum_{\tau=t}^{T-1} \left(F^\top\right)^{\tau-t} P\left(\theta_\tau s_\tau - \kappa(\tau)\widetilde{\theta}_t \left(\lambda \circ \widetilde{s}_{\tau|t}\right)\right)\right), \tag{22}$$

where $\kappa(\tau) \equiv 0$ when $\tau > \overline{T}$. The sum of quadratic costs in (22) can be further bounded by

$$J(\lambda\text{-Con}) - J^\star$$
$$\leq \|H\| \sum_{t=0}^{T-1} \left\| \sum_{\tau=t}^{T-1} \left(F^\top\right)^{\tau-t} P\left(\theta s_\tau - \kappa(\tau)\widetilde{\theta}_t \left(\lambda \circ \widetilde{s}_{\tau|t}\right)\right) \right\|^2$$
$$\leq 2\|H\| \sum_{t=0}^{T-1} \left( \underbrace{\left\| \sum_{\tau=t}^{\overline{T}} \left(F^\top\right)^{\tau-t} P\left(\theta s_\tau - \widetilde{\theta}_t \left(\lambda \circ \widetilde{s}_{\tau|t}\right)\right) \right\|^2}_{\text{short-term error } g_t^{\text{error}} \text{ (linear setting)}} + \underbrace{\left\| \sum_{\tau=t+w}^{T-1} \left(F^\top\right)^{\tau-t} P\theta s_\tau \right\|^2}_{\text{long-term error } h_t^{\text{error}} \text{ (linear setting)}} \right). \tag{23}$$

In above, the first term characterizes a total cost gap induced by inaccurate ICA identifiability and predictions. The second term in (23) is the long-term error for disentangled times series predictions that are at least $w$ steps away from the current time steps, denoted by $h_t^{\text{error}}$, which becomes 0 when $t + w \geq T - 1$. It follows that

$$h_t^{\text{error}} \leq \left( \sum_{\tau=t+w}^{T-1} C_F \rho_F^{\tau-t} \|P\| \overline{\theta}\overline{s} \right)^2 \leq \left( \frac{C_F}{1-\rho_F} \|P\| \overline{\theta}\overline{s} \right)^2 \rho_F^{2w}, \tag{24}$$

since by our assumption on $F$, the Gelfand's formula implies that there must exist a constant $C_F > 0$, $\rho_F \in (0,1)$ s.t. $\|F^t\| \le C_F \rho_F^t$ for all $t \ge 0$. Moreover, for $g_t^{\text{error}}$, we can bound it from above by

$$g_t^{\text{error}} = \left\| \sum_{\tau=t}^{\overline{T}} \left(F^\top\right)^{\tau-t} P\left(\theta s_\tau - \widetilde{\theta}_t s_\tau + \widetilde{\theta}_t s_\tau - \widetilde{\theta}_t \left(\lambda \circ \widetilde{s}_{\tau|t}\right)\right) \right\|^2$$

$$\le 2 \left\| \sum_{\tau=t}^{\overline{T}} \left(F^\top\right)^{\tau-t} P\eta_t s_\tau \right\|^2 + 2 \left\| \sum_{\tau=t}^{\overline{T}} \left(F^\top\right)^{\tau-t} P\left(\widetilde{\theta}_t \left(s_\tau - \lambda \circ \widetilde{s}_{\tau|t}\right)\right) \right\|^2, \qquad (25)$$

where $\eta_t := \widetilde{\theta}_t - \theta$ is the mixing parameter error in (4) defined Section 2.2. Furthermore, noting $\left(\varepsilon_{\tau|t}(1), \dots, \varepsilon_{\tau|t}(k)\right) = \varepsilon_{\tau|t} := s_\tau - \widetilde{s}_{\tau|t}$, we obtain

$$g_t^{\text{error}} \le 2 \left\| \sum_{\tau=t}^{\overline{T}} \left(F^\top\right)^{\tau-t} P\eta_t s_\tau \right\|^2 + 2 \left\| \sum_{\tau=t}^{\overline{T}} \left(F^\top\right)^{\tau-t} P\left(\widetilde{\theta}_t \left(\lambda \circ \varepsilon_{\tau|t} + (\mathbf{1}_k - \lambda) \circ s_\tau\right)\right) \right\|^2$$

$$\le 2 \left( \sum_{\tau=t}^{\overline{T}} C_F \rho_F^{\tau-t} \|P\eta_t s_\tau\| \right)^2 + 4 \underbrace{\left\| \sum_{\tau=t}^{\overline{T}} \left(F^\top\right)^{\tau-t} P\left(\widetilde{\theta}_t \left(\lambda \circ \varepsilon_{\tau|t}\right)\right) \right\|^2}_{\text{consistency error } g_t^{\text{con}} \text{ (ICA setting)}}$$

$$+ 4 \underbrace{\left\| \sum_{\tau=t}^{\overline{T}} \left(F^\top\right)^{\tau-t} P\left(\widetilde{\theta}_t \left((\mathbf{1}_k - \lambda) \circ s_\tau\right)\right) \right\|^2}_{\text{robustness error } g_t^{\text{rob}} \text{ (ICA setting)}}. \qquad (26)$$

The right hand side of (26) contains two types of errors - the consistency error, denoted by $g_t^{\text{con}}$ that occurs due to the estimated time series; and the robustness error, denoted by $g^{\text{rob}}(t)$ that arises when $\lambda$ becomes small. Therefore, combining (23), (24), and (26) implies the following upper bound on $J(\lambda\text{-CON}) - J^\star$:

$$J(\lambda\text{-CON}) - J^\star \le 2\|H\| \sum_{t=0}^{T-1} \left(g_t^{\text{error}} + h_t^{\text{error}}\right)$$

$$\le 4\|H\| \left(\frac{C_F}{1-\rho_F} \|P\|\overline{s}\right)^2 \overline{\eta} + 2\|H\|T \left(\frac{C_F}{1-\rho_F} \|P\|\overline{\theta}\overline{s}\right)^2 \rho_F^{2w} + 8\|H\| \sum_{t=0}^{T-1} \left(g_t^{\text{con}} + g_t^{\text{rob}}\right),$$

where $\overline{\eta}$ is defined in (4).

$\square$

**Step 2: Component-Wise Consistency Bound on $g_t^{\text{con}}$**   In the sequel, we bound $g_t^{\text{con}}$ and $g^{\text{rob}}(t)$ respectively. First, the consistency error caused by inaccurate time series predictions can be bounded by

$$g_t^{\text{con}} \le \left( \sum_{\tau=t}^{\overline{T}} C_F \rho_F^{\tau-t} \overline{\theta}\|P\| \left\|\lambda \circ \varepsilon_{\tau|t}\right\| \right)^2 \le w \left(C_F\|P\|\overline{\theta}\right)^2 \sum_{\tau=t}^{\overline{T}} \rho_F^{2(\tau-t)} \left\|\lambda \circ \varepsilon_{\tau|t}\right\|^2,$$

which can be further decomposed into a summation of component-wise latent variable time-series prediction errors by denoting $\overline{\varepsilon}(i) := \sum_{t=0}^{T-1} \sum_{\tau=t}^{T} (\rho_F^{(\tau-t)} \varepsilon_{\tau|t}(i))^2$ where $\varepsilon_{\tau|t} = (\varepsilon_{\tau|t}(i) : i =$

$1, \ldots, k)$, as defined in Section 2.2. Considering $\lambda = (\lambda(1), \ldots, \lambda(k))$, it follows that

$$
\begin{aligned}
\sum_{t=0}^{T-1} g_t^{\text{con}} &\leq w \left( C_F \|P\|\overline{\theta} \right)^2 \sum_{t=0}^{T-1} \sum_{\tau=t}^{\overline{T}} \rho_F^{2(\tau-t)} \left\| \lambda \circ \varepsilon_{\tau|t} \right\|^2 \\
&\leq w \left( C_F \|P\|\overline{\theta} \right)^2 \sum_{t=0}^{T-1} \sum_{\tau=t}^{\overline{T}} \rho_F^{2(\tau-t)} \left\| \lambda \circ \varepsilon_{\tau|t} \right\|^2 \\
&= w \left( C_F \|P\|\overline{\theta} \right)^2 \sum_{i=1}^{k} (\lambda(i))^2 \, \overline{\varepsilon}(i).
\end{aligned}
\tag{27}
$$

**Step 3: Robustness bound on $g_t^{\text{rob}}$**  Next, we consider deriving a bound on $g_t^{\text{rob}}$. We obatin

$$
g_t^{\text{rob}} = \left\| \sum_{\tau=t}^{\overline{T}} \left( F^\top \right)^{\tau-t} P \left( \widetilde{\theta}_\tau \left( (\mathbf{1}_k - \lambda) \circ s_\tau \right) \right) \right\|^2 \leq \left( C_F \|P\|\overline{\theta} \right)^2 \left( \sum_{\tau=t}^{\overline{T}} \rho_F^{\tau-t} \left\| (\mathbf{1}_k - \lambda) \circ s_\tau \right\| \right)^2,
$$

which leads to

$$
g_t^{\text{rob}} \leq \left( C_F \|P\|\overline{\theta} \right)^2 \left\| (\mathbf{1}_k - \lambda) \right\|_4^2 \left( \sum_{\tau=t}^{\overline{T}} \rho_F^{\tau-t} \|s_\tau\|_4 \right)^2
\tag{28}
$$

by applying the Cauchy–Schwarz inequality where $\|\cdot\|_4$ denotes the $\ell_4$-norm. Denote by $\sigma_{\min}(\theta) > 0$ the smallest singular value of the full column rank mixing matrix $\theta$. Noting that $\|s_\tau\|_4 \leq \|s_\tau\| \leq \frac{1}{\sigma_{\min}(\theta)} \|\theta s_\tau\|$ for all $\tau$, and $f(s; \theta) = \theta s$ in this linear mixing setting, applying Lemma 2,

$$
\frac{1}{J^\star} \sum_{t=0}^{T-1} g_t^{\text{rob}} \leq \frac{\left( C_F \|P\|\overline{\theta} \right)^2 \left\| (\mathbf{1}_k - \lambda) \right\|_4^2}{C_0 \sigma_{\min}^2(\theta)}
\tag{29}
$$

for some constant $0 < C_0 \leq \frac{(1-\rho_F)^2}{2} \min\{\lambda_{\min}(P), \lambda_{\min}(R)/\|B\|, \lambda_{\min}(Q)/\max\{2, \|A\|\}\}$.

**Step 4: Competitive Analysis of $\lambda$-CON**  Putting together Lemma 4, (27), and (29), since $J^\star > 0$,

$$
\frac{J(\lambda\text{-CON}) - J^\star}{J^\star} \leq \frac{1}{J^\star} \left( 2\|H\| \left( 2\|P\|C_f \overline{s} \right)^2 \left( \overline{\eta} + \left( \overline{\theta}\rho_F^w \right)^2 T \right) \right) + \frac{8\|H\|}{J^\star} \sum_{t=0}^{T-1} \left( g_t^{\text{con}} + g_t^{\text{rob}} \right),
\tag{30}
$$

where the last term can be further bounded from above by

$$
\begin{aligned}
\frac{8\|H\|}{J^\star} \sum_{t=0}^{T-1} \left( g_t^{\text{con}} + g_t^{\text{rob}} \right) &\leq 8\|H\| \left( C_F \|P\|\overline{\theta} \right)^2 \left( \frac{w}{J^\star} \sum_{i=1}^{k} (\lambda(i))^2 \, \overline{\varepsilon}(i) + \frac{\left\| (\mathbf{1}_k - \lambda) \right\|_4^2}{C_0 \sigma_{\min}^2(\theta)} \right) \\
&\leq 8\|H\| \left( C_F \|P\|\overline{\theta} \right)^2 \left( \sum_{i=1}^{k} \left( \frac{w\overline{\varepsilon}(i)}{J^\star} (\lambda(i))^2 + \frac{(1-\lambda(i))^2}{C_0 \sigma_{\min}^2(\theta)} \right) \right).
\end{aligned}
\tag{31}
$$

**Step 5: Online Learning of $\lambda_t$ via FTRL**  Consider a follow-the-regularized-leader (FTRL) optimization with an $\ell_2$-regularizer $\|\lambda - \lambda_0\|^2$. The following lemma can be proved based on the equivalent online convex optimization (OCO) form of the online mirror descent (OMD) (see the implementation described in Section 5) above as shown in Lemma 1. We reprise the OMD steps below.

$$
\tau_t = \tau_{t-1} - \frac{\beta}{2} \nabla_\lambda \left( \zeta_t \right),
\tag{32a}
$$

$$
\lambda_t = \text{Proj}_{\mathcal{I}}(\tau_t).
\tag{32b}
$$

Now, fix $\lambda = (\lambda_t : t \in [T])$ generated by DISC. Let us consider a pseudo-cost $\Delta J^\dagger(\lambda)$ defined below, which is compared with the true objective $\Delta J(\lambda)$ of DISC.

$$\Delta J^{\dagger}(\lambda) := \sum_{\ell=0}^{T-1} \sum_{\tau=\underline{\ell}}^{\ell} \left[ (F^{\top})^{\ell-\tau} P \left( \theta s_{\ell} - \widetilde{\theta}_{\tau} (\lambda_{\ell} \circ \widetilde{s}_{\ell|\tau}) \right) \right]^{\top} H \left[ (F^{\top})^{\ell-\tau} P \left( \theta s_{\ell} - \widetilde{\theta}_{\tau} (\lambda_{\ell} \circ \widetilde{s}_{\ell|\tau}) \right) \right]$$

$$\Delta J(\lambda) = \sum_{\ell=0}^{T-1} \sum_{\tau=\underline{\ell}}^{\ell} \left[ (F^{\top})^{\ell-\tau} P \left( \theta s_{\ell} - \widetilde{\theta}_{\tau} (\lambda_{\tau} \circ \widetilde{s}_{\ell|\tau}) \right) \right]^{\top} H \left[ (F^{\top})^{\ell-\tau} P \left( \theta s_{\ell} - \widetilde{\theta}_{\tau} (\lambda_{\tau} \circ \widetilde{s}_{\ell|\tau}) \right) \right]$$

The lemma in the following holds.

**Lemma 5.** *The online mirror descent (OMD)* (32a)-(32b) *generates a sequence* $\lambda = (\lambda_0, \ldots, \lambda_{T-1})$ *in* DISC *such that*

$$\Delta J^{\dagger} - \min_{\lambda \in \mathcal{I}} \Delta J(\lambda) \leq 8 \|H\| \left( \overline{s}\overline{\theta} \|P\| \frac{C_f}{1 - \rho_F} \right)^2 \sqrt{kT}.$$

*Proof of Lemma 5.* The gradient of the cost gap at time $t \in [T]$ satisfies

$$\nabla_{\lambda}(\zeta_t) = \sum_{t=0}^{T-1} \sum_{\tau=\underline{\ell}}^{\ell} \mathsf{J}^{\top}(\tau, t) \left( H + H^{\top} \right) g(\tau, t)$$

$$= 2 \sum_{t=0}^{T-1} \sum_{\tau=\underline{\ell}}^{\ell} \mathsf{J}^{\top}(\tau, t) H \left( F^{\top} \right)^{t-\tau} P \left( \theta s_t - \widetilde{\theta}_{\tau} \left( \lambda \circ \widetilde{s}_{t|\tau} \right) \right), \tag{33}$$

where in (33) we have used the fact that $H := B(R + B^{\top}PB)^{-1}B^{\top}$ is symmetric. Moreover, noting that $\widetilde{s}_{t|\tau} = \left( \widetilde{s}_{t|\tau}(1), \ldots, \widetilde{s}_{t|\tau}(k) \right)$, in (33) above,

$$\mathsf{J}(\tau, t) := - \left( F^{\top} \right)^{t-\tau} P \widetilde{\theta}_{\tau} \begin{bmatrix} \widetilde{s}_{t|\tau}(1) & & \\ & \ddots & \\ & & \widetilde{s}_{t|\tau}(k) \end{bmatrix}$$

denotes an $n \times k$ Jacobian matrix $\mathsf{J}(\tau, t)$ of $g(\tau, t)$ with respect to $\lambda$ whose $(i, j)$-th entry is $\frac{\partial g(\tau, t)^{(i)}}{\partial \lambda^{(j)}}$. Normalizing the predicted time series yields that

$$\|\nabla_{\lambda}(\zeta_t)\| \leq 4 \|H\| \left( \overline{s}\overline{\theta} \|P\| \frac{C_f}{1 - \rho_F} \right)^2.$$

As we have verified in Appendix D.1, each $g(\tau, t)^{\top}(\lambda) H g(\tau, t)$ is a convex function of $\lambda \in \mathcal{I}$. Therefore, based on the static regret analysis for OCO [23, 76], FTRL with an $\ell_2$-regularizer guarantees (after optimizing $\beta$) that

$$\Delta J^{\dagger}(\lambda) - \min_{\lambda \in \mathcal{I}} \Delta J(\lambda)$$

$$\leq 2 \|\nabla_{\lambda}(\zeta_t)\| \max_{\lambda \in \mathcal{I}} \|\lambda - \lambda_0\| \sqrt{T}$$

$$\leq 8 \|H\| \left( \overline{s}\overline{\theta} \|P\| \frac{C_f}{1 - \rho_F} \right)^2 \sqrt{kT}. \tag{34}$$

$\square$

Applying Assumption 2 and noting that $\lambda_{\ell}^2 - \lambda_{\ell-\tau}^2 \leq O(|\lambda_{\ell} - \lambda_{\ell-\tau}|)$ for all $\ell \in [T]$, we conclude that $\Delta J(\lambda) - \Delta J^{\dagger}(\lambda) = o(T)$, since for $\tau$ that is not a constant, the exponent $\rho^{\tau} = o(1)$.

Denote by $\lambda^*$ the optimal solution that minimizes $J(\lambda\text{-CON})$. Using 30 and (34) above, we conclude that

$$\frac{J(\text{DISC})}{J^{\star}} = 1 + \underbrace{\frac{J(\lambda^*\text{-CON}) - J^{\star}}{J^{\star}}}_{\text{Lemma 4 and (31)}} + \underbrace{\frac{J(\text{DISC}) - J(\lambda^*\text{-CON})}{J^{\star}}}_{\text{Lemma 5}},$$

whose right hand side can be bounded respectively via Lemma 4 and Lemma 5. Furthermore, noting that $f(s_t; \theta) \neq 0$ for all $t \in [T]$, hence $\|\theta s_t\| > 0$. As a result, Lemma 2 implies $J^\star = \Omega(T)$. Therefore, using Lemma 1,

$$
\begin{aligned}
\frac{J(\textsc{Disc})}{J^\star} \leq\ &1 + 2\|H\| \left(2\|P\|\frac{C_f}{1-\rho_F}\overline{s}\right)^2 \frac{1}{J^\star} \left(\overline{\eta} + \left(\overline{\theta}\rho_F^w\right)^2 T\right) \\
&+ 8\|H\| \left(C_F\|P\|\overline{\theta}\right)^2 \min_{\lambda \in \mathcal{I}} \left(\sum_{i=1}^{k} \left(\frac{w\overline{\varepsilon}(i)}{J^\star}(\lambda(i))^2 + \frac{(1-\lambda(i))^2}{C_0\sigma_{\min}^2(\theta)}\right)\right) \\
&+ 8\|H\| \left(\|P\|\frac{C_f}{1-\rho_F}\overline{s}\overline{\theta}\right)^2 \frac{w\sqrt{kT}}{J^\star} + o(1).
\end{aligned}
$$

After optimizing over $\lambda \in \mathcal{I}$,

$$
\begin{aligned}
\frac{J(\textsc{Disc})}{J^\star} \leq\ &1 + 2\|H\| \left(2\|P\|C_f\overline{s}\right)^2 \frac{1}{J^\star} \left(\overline{\eta} + \left(\overline{\theta}\rho_F^w\right)^2 T\right) \\
&+ 8\|H\| \left(C_F\|P\|\overline{\theta}\right)^2 \left(\sum_{i=1}^{k} \left(\frac{w\overline{\varepsilon}(i)}{\sigma_{\min}^2(\theta)C_0 w\overline{\varepsilon}(i) + J^\star}\right)\right) \\
&+ 8\|H\| \left(\|P\|\frac{C_f}{1-\rho_F}\overline{s}\overline{\theta}\right)^2 \frac{w\sqrt{kT}}{J^\star} + o(1).
\end{aligned}
$$

## F   Proof of Corollary B.1

Under the classic assumptions such that $\mathbb{E}\left[s_t(i)\right] = 0$, $\mathbb{E}\left[s_t^2(i)\right] = 1$, $\mathbb{E}\left[\left|s_t^5(i)\right|\right] \leq M$, and the fourth cumulant $|\mathrm{cum}_4(s_t(i))| \geq \Delta$ for all $t \in [T]$ and $i = 1, \ldots, k$, the Recursive Fourier PCA algorithm in [75] is a polynomial-time and it guarantees (c.f. Theorem 1 in [75]) that for all $t \in [T]$,

$$
\|\eta_t\| = \|\theta - \theta_t\| \leq \|\theta - \theta_t\|_F \leq \frac{\overline{\theta}\overline{s}(\log n)^{7/2}M^{1/2}k^{1/2}}{\Delta^3 t^{1/2}},
$$

with high probability, which yields $\overline{\eta} = O\left((\log n)^{7/2}k^{1/2}\sum_{t\in[T]}\frac{1}{t}\right) = O\left((\log n)^{7/2}k^{1/2}\log T\right)$. Therefore, with high probability, the competitive ratio bound in the corollary holds.

## G   Proof of Theorem 4.1

Denote $\overline{T} := \min\{\ell + w - 1, T - 1\}$. Similar to the analysis in the proof of Theorem 4.2 in Appendix E, we obtain

$$
\begin{aligned}
&J\left(\lambda\text{-}\textsc{Con}\right) - J^\star \\
&= \sum_{\ell=0}^{T-1} \left(\sum_{\tau=\ell}^{\overline{T}} \left(F^\top\right)^{\tau-\ell} P\left(f_\tau - \widetilde{f}_{\tau|\ell}(\lambda)\right)\right)^\top H \left(\sum_{\tau=\ell}^{\overline{T}} \left(F^\top\right)^{\tau-\ell} P\left(f_\tau - \widetilde{f}_{\tau|\ell}(\lambda)\right)\right).
\end{aligned}
\tag{35}
$$

WLOG, we assume $f$ is linear such that $f(s; \theta) = \theta s$ for some $n \times k$ matrix $\theta = (\theta_{ij})$ and the estimated mixing matrix is accurate so that $\overline{\eta} = 0$. Then it follows that

$$
J\left(\lambda\text{-}\textsc{Con}\right) - J^\star \geq \lambda_{\min}\left((R + B^\top PB)^{-1}\right) \sum_{\ell=0}^{T-1} \left\|B^\top \psi_{\ell,T}\right\|^2,
$$

where

$$
\psi_{\ell,T} := \sum_{\tau=\ell}^{\overline{T}} \left(F^\top\right)^{\tau-\ell} P\theta_\tau \left(\lambda \circ \varepsilon_{\tau|\ell} - (\mathbf{1}_k - \lambda) \circ s_{\tau|\ell}\right).
\tag{36}
$$

Since $\lambda$-Con is $(1 + o(1))$-consistent, we must have $J(\lambda\text{-Con}) - J^\star \leq o(1)J^\star$ (by our model assumption that $s_t \neq 0$ for all $t \in [T]$, $J^\star > 0$). Combining this with (36) above, it is necessary to have

$$o(1)J^\star \geq \lambda_{\min}\left((R + B^\top PB)^{-1}\right) \sum_{t=0}^{T-1} \left\| B^\top \sum_{\tau=\ell}^{\overline{T}} \left(F^\top\right)^{\tau-\ell} P\theta_\tau\left((\mathbf{1}_k - \lambda) \circ s_{\tau|\ell}\right) \right\|^2. \quad (37)$$

The competitive ratio is the worst-case ratio for all $A, B, Q, R, f, \theta$, and $(s_t : t \in [T])$, we must have $\lambda = \mathbf{1}_k$ to guarantee (37) above.

Now, consider an adaptive offline adversary that generates $\varepsilon_{\tau|\ell} = \mu s_{\tau|\ell}$ for all $\ell \leq \tau \leq \overline{\ell}$. Since $\lambda = \mathbf{1}_k$, we obtain

$$J(\lambda\text{-Con}) - J^\star \geq \mu^2 \lambda_{\min}\left((R + B^\top PB)^{-1}\right) \sum_{t=0}^{T-1} \left\| B^\top \sum_{\tau=\ell}^{\overline{T}} \left(F^\top\right)^{\tau-\ell} P\theta_\tau s_{\tau|\ell} \right\|^2.$$

Setting $\mu = \log T$, above implies $\mathsf{CR}(\lambda\text{-Con}) = \omega(1)$.

# H    Proof of Theorem 4.3

We first state a detailed version of Theorem 4.2 below with explicit multiplicative constants.

 **Step 1: Consistency-Robustness Analysis of $\lambda$-Con**    We first show the following lemma, which highlights a tradeoff between consistency and robustness.

**Lemma 6.** *With a fixed trust parameter $\lambda \in \mathcal{I}$, the disentangled $\lambda$-confident control (DISC) has a worst-case dynamic regret of at most*

$$4\|H\|\left(\frac{C_F}{1 - \rho_F}C_f\|P\|\right)^2 \overline{\eta} + 2\|H\|T\left(\frac{C_F}{1 - \rho_F}C_f\|P\|f_{\max}\right)^2 \rho_F^{2w} + 8\|H\|\sum_{t=0}^{T-1}\left(g_t^{con} + g_t^{rob}\right) \tag{38}$$

*where $H := B(R + B^\top PB)^{-1}B^\top$; $f_{\max} := f_0 + C_f\overline{s}$ with $f_0 := |f(0)|$; $P, C_F, \rho_F, w$ are defined in Section 2; $C_f$ denotes the Lipschitz constant of $f$. The terms $g_t^{con}$ and $g_t^{rob}$ are functions of $\lambda$, defined as*

$$g_t^{con}(\lambda) := \left(\sum_{\tau=t}^{\overline{T}} C_F C_f \rho_F^{\tau-t}\|P\|\,\|\lambda \circ \varepsilon_{\tau|t}\|\right)^2,$$

$$g_t^{rob}(\lambda) := \left(\sum_{\tau=t}^{\overline{T}} C_F C_f \rho_F^{\tau-t}\|P\|\,\|(\mathbf{1}_k - \lambda) \circ s_\tau\|\right)^2.$$

*Proof of Lemma 6.* Similar to the analysis in the proof of Theorem 4.2 in Appendix E, we obtain

$$J(\lambda\text{-Con}) - J^\star$$
$$= \sum_{t=0}^{T-1}\left(\sum_{\tau=t}^{T-1}\left(F^\top\right)^{\tau-t}P\left(f_\tau - \widetilde{f}_{\tau|t}(\lambda)\right)\right)^\top H\left(\sum_{\tau=t}^{T-1}\left(F^\top\right)^{\tau-t}P\left(f_\tau - \widetilde{f}_{\tau|t}(\lambda)\right)\right) \tag{39}$$

where we simplify the latent perturbations and the corresponding predictions as $f_\tau := f(s_\tau; \theta)$ and $\widetilde{f}_{\tau|t}(\lambda) := f\left(\lambda \circ \widetilde{s}_{\tau|t}; \widetilde{\theta}_t\right)$. Especially, $\widetilde{f}_{\tau|t}(\lambda) := 0$ when $\tau > \overline{T}$. Thus, the total cost gap can be

bounded by

$$J\left(\lambda\text{-CON}\right) - J^\star$$

$$\leq \|H\| \sum_{t=0}^{T-1} \Big\| \sum_{\tau=t}^{T-1} \left(F^\top\right)^{\tau-t} P \left(f_\tau - \widetilde{f}_{\tau|t}(\lambda)\right) \Big\|^2$$

$$\leq 2\|H\| \sum_{t=0}^{T-1} \Big( \underbrace{\Big\| \sum_{\tau=t}^{\overline{T}} \left(F^\top\right)^{\tau-t} P \left(f_\tau - \widetilde{f}_{\tau|t}(\lambda)\right) \Big\|^2}_{\text{short-term error } g_t^{\text{error}} \text{ (general mixing)}} + \underbrace{\Big\| \sum_{\tau=t+w}^{T-1} \left(F^\top\right)^{\tau-t} P f_\tau \Big\|^2}_{\text{long-term error } h_t^{\text{error}} \text{ (general mixing)}} \Big). \qquad (40)$$

Note that by Assumption 1 $f$ is continuous with a Lipschitz constant $C_f > 0$, hence, $f$ is bounded, since $\|s_\tau\| \leq \overline{s}$. Denote by $f_{\max} \leq f_0 + C_f \overline{s}$ the largest value of $f$ where $f_0 := |f(0)|$. The second term $h_t^{\text{error}}$ in (23) can be bounded similarly to (24) in the proof of Theorem 4.2 (see Appendix E):

$$h_t^{\text{error}} \leq \left( \sum_{\tau=t+w}^{T-1} C_F \rho_F^{\tau-t} C_f \|P\| f_{\max} \right)^2 \leq \left( \frac{C_F}{1-\rho_F} C_f \|P\| f_{\max} \right)^2 \rho_F^{2w}, \qquad (41)$$

The first term $g_t^{\text{error}}$ in (23) characterizes a total cost gap induced by inaccrurate mixing matrix estimation and time series prediction errors. It follows that

$$g_t^{\text{error}} = \Bigg\| \sum_{\tau=t}^{\overline{T}} \left(F^\top\right)^{\tau-t} P \left( f\left(s_\tau;\theta\right) - f\left(s_\tau;\widetilde{\theta}_t\right) + f\left(s_\tau;\widetilde{\theta}_t\right) - f\left(\lambda \circ \widetilde{s}_{\tau|t};\widetilde{\theta}_t\right) \right) \Bigg\|^2$$

$$\leq 2 \left( \sum_{\tau=t}^{\overline{T}} C_F C_f \rho_F^{\tau-t} \|P\| \left\|\theta - \widetilde{\theta}_t\right\| \right)^2 + 2 \left( \sum_{\tau=t}^{\overline{T}} C_F C_f \rho_F^{\tau-t} \|P\| \left\|s_\tau - \lambda \circ \widetilde{s}_{\tau|t}\right\| \right)^2, \qquad (42)$$

where we have used the fact that $\|F^t\| \leq C_F \rho_F^t$ for all $t \in [T]$ with a spectral radius $\rho_F \in (0,1)$.

Recall that $\eta_t := \widetilde{\theta}_t - \theta$ is the mixing parameter error in (4) defined Section 2.2. Furthermore, noting $\left(\varepsilon_{\tau|t}(1), \ldots, \varepsilon_{\tau|t}(k)\right) = \varepsilon_{\tau|t} := s_\tau - \widetilde{s}_{\tau|t}$, continuing from (42), we get

$$g_t^{\text{error}} \leq 2 \left( \sum_{\tau=t}^{\overline{T}} C_F C_f \rho_F^{\tau-t} \|P\| \eta_t \right)^2 + 2 \left( \sum_{\tau=t}^{\overline{T}} C_F C_f \rho_F^{\tau-t} \|P\| \left\|s_\tau - \lambda \circ \widetilde{s}_{\tau|t}\right\| \right)^2$$

$$\leq 2 \left( \sum_{\tau=t}^{\overline{T}} C_F C_f \rho_F^{\tau-t} \|P\| \eta_t \right)^2 + 4 \underbrace{\left( \sum_{\tau=t}^{\overline{T}} C_F C_f \rho_F^{\tau-t} \|P\| \left\|\lambda \circ \varepsilon_{\tau|t}\right\| \right)^2}_{\text{consistency error } g_t^{\text{con}} \text{ (general mixing)}}$$

$$+ 4 \underbrace{\left( \sum_{\tau=t}^{\overline{T}} C_F C_f \rho_F^{\tau-t} \|P\| \left\|(\mathbf{1}_k - \lambda) \circ s_\tau\right\| \right)^2}_{\text{robustness error } g_t^{\text{rob}} \text{ (general mixing)}}. \qquad (43)$$

The right hand side of (43) contains two types of errors - the consistency error, denoted by $g_t^{\text{con}}$ that occurs due to the estimated time series; and the robustness error, denoted by $g^{\text{rob}}(t)$ that arises when $\lambda$ becomes small. Therefore, rearranging the terms and combining (40), (41), and (43) imply the following upper bound on $J\left(\lambda\text{-CON}\right) - J^\star$:

$$J\left(\lambda\text{-CON}\right) - J^\star \leq 2\|H\| \sum_{t=0}^{T-1} \left(g_t^{\text{error}} + h_t^{\text{error}}\right)$$

$$\leq 4\|H\| \left( \frac{C_F}{1-\rho_F} C_f \|P\| \right)^2 \overline{\eta} + 2\|H\| T \left( \frac{C_F}{1-\rho_F} C_f \|P\| f_{\max} \right)^2 \rho_F^{2w} + 8\|H\| \sum_{t=0}^{T-1} \left(g_t^{\text{con}} + g_t^{\text{rob}}\right),$$

$$(44)$$

with $\overline{\eta}$ defined in (4). $\qquad\qquad \square$

Furthermore, we can bound the terms $g_t^{\text{error}}$ and $h_t^{\text{error}}$ separately as:

$$\sum_{t=0}^{T-1} g_t^{\text{con}} \leq w \left( C_F C_f \|P\| \right)^2 \sum_{i=1}^{k} \left( \lambda(i) \right)^2 \bar{\varepsilon}(i), \tag{45}$$

and

$$\frac{1}{J^\star} \sum_{t=0}^{T-1} g_t^{\text{rob}} \leq \frac{\left( C_F C_f \|P\| \right)^2 \|(\mathbf{1}_k - \lambda)\|_4^2}{C_0 \|f^{-1}\|^2}, \tag{46}$$

where $f^{-1}$ is the inverse function of the mixing function $f$. Since by Assumption 1, $f$ is bijective, $f^{-1}$ must exist. Thus, (45) and (46) together imply

$$\frac{1}{J^\star} \sum_{t=0}^{T-1} \left( g_t^{\text{con}} + g_t^{\text{rob}} \right) \leq \left( C_F C_f \|P\| \right)^2 \left( \sum_{i=1}^{k} \left( \frac{w\bar{\varepsilon}(i)}{J^\star} \left( \lambda(i) \right)^2 + \frac{(1-\lambda(i))^2}{C_0 \|f^{-1}\|^2} \right) \right). \tag{47}$$

**Step 2: Online Learning of $\lambda_t$ via FTPL**   Consider a follow-the-perturbed-leader (FTPL) optimization [28, 25], with:

$$\lambda_t \in \operatorname*{arg\,min}_{\lambda \in \mathcal{I}} \left( \sum_{\ell=0}^{t-1} \zeta_\ell(\lambda) + \sigma_t^\top \lambda \right) \quad \text{(FTPL FOR $\lambda$-LEARNING)} \tag{48}$$

The following lemma is a result of [25] by fixing each coordinate $\sigma_t(i)$ $(i = 1, \ldots, k)$ an IID random variable from an exponential distribution with some parameter $\alpha > 0$.

**Lemma 7.** *The FTPL procedure* (48) *above generates a sequence* $\lambda = (\lambda_0, \ldots, \lambda_{T-1})$ *in* DISC *such that*

$$\mathbb{E} \left[ \frac{1}{T} \left( \Delta J^\dagger(\lambda) - \min_{\lambda \in \mathcal{I}} \Delta J\left( (\lambda) \right) \right) \right] \leq O \left( \alpha k^{5/2} C_f^2 + \frac{k^{3/2}}{\alpha T} \right). \tag{49}$$

Similarly to the proof of Theorem 4.2, applying Assumption 2, $J(\text{DISC}) - J^\dagger = o(T)$. Denote by $\lambda^*$ the optimal solution that minimizes $J(\lambda\text{-CON})$. Putting together (44), (47), and (49), we conclude

$$\mathbb{E} \left[ \frac{J(\text{DISC})}{J^\star} \right] = 1 + \underbrace{\mathbb{E} \left[ \frac{J(\lambda^*\text{-CON}) - J^\star}{J^\star} \right]}_{\text{Lemma 6 and Eq. (47)}} + \underbrace{\mathbb{E} \left[ \frac{J(\text{DISC}) - J(\lambda^*\text{-CON})}{J^\star} \right]}_{\text{Lemma 7}},$$

yielding the following after optimizing over $\alpha > 0$:

$$\mathbb{E} \left[ \frac{J(\text{DISC})}{J^\star} \right] \leq 1 + \frac{C_3}{J^\star} \left( 4\bar{\eta} + 2T \left( f_{\max} \rho_F^w \right)^2 \right) + 8C_4 \min_{\lambda \in \mathcal{I}} \left( \sum_{i=1}^{k} \left( \frac{w\bar{\varepsilon}(i)}{J^\star} \left( \lambda(i) \right)^2 + \frac{(1-\lambda(i))^2}{C_0 \|f^{-1}\|^2} \right) \right)$$
$$+ O \left( k^2 C_f \frac{w\sqrt{T}}{J^\star} \right) \tag{50}$$
$$\leq 1 + O \left( \rho_F^{2w} + \frac{\bar{\eta}}{T} \right) + 8C_4 \left( \sum_{i=1}^{k} \left( \frac{w\bar{\varepsilon}(i)}{C_0 \|f^{-1}\|^2 w\bar{\varepsilon}(i) + J^\star} \right) \right) + O \left( \frac{wk^2}{\sqrt{T}} \right), \tag{51}$$

where we denote $C_3 := \|H\| \left( \frac{C_F}{1-\rho_F} C_f \|P\| \right)^2$ and $C_4 := \|H\| \left( C_F C_f \|P\| \right)^2$ and have used $J^\star = \Omega(T)$ in (50) (implied by Lemma 2 and the assumption that $f$ is non-zero and Lipschitz continuous, similar to the proof of Theorem 4.2 in Appendix E) to derive (51).

